# Efficient Reinforcement Learning by Discovering Neural Pathways

**Samin Yeasar Arnob**
Department of Computer Science
McGill University
Mila Quebec AI Institute
samin.arnob@mail.mcgill.ca

**Riyasat Ohib**
Georgia Institute of Technology

**Sergey Plis**
Georgia State University

**Amy Zhang**
University of Texas, Austin

**Alessandro Sordoni**
Microsoft Research

**Doina Precup**
McGill University
Mila Quebec AI Institute

## Abstract

Reinforcement learning (RL) algorithms have been very successful at tackling complex control problems, such as AlphaGo or fusion control. However, current research mainly emphasizes solution quality, often achieved by using large models trained on large amounts of data, and does not account for the financial, environmental, and societal costs associated with developing and deploying such models. Modern neural networks are often overparameterized and a significant number of parameters can be pruned without meaningful loss in performance, resulting in more efficient use of the model's capacity. We present a methodology for identifying sparse sub-networks within a larger network in reinforcement learning (RL). We call such sub-networks *neural pathways*. We show empirically that even very small learned sub-networks, using less than 5% of the large network's parameters, can provide very good quality solutions. We also demonstrate the training of multiple pathways within the same networks in a multi-task setup, where each pathway tackles a separate task. We evaluate empirically our approach on several continuous control tasks, in both online and offline settings.

## 1 Introduction

Scaling large neural-network models [4, 73, 33, 9, 12, 76, 64] has led to state of the art in machine learning benchmarks. While the utility of these large models in a variety of applications makes them compelling for widespread use, it also requires very expensive training with a large carbon footprint [78, 71, 87, 20, 88, 79, 87]. While the human brain serves as an inspiration for deep learning techniques, the current deep learning architectures do not exhibit the same level of energy efficiency. The brain continuously learns new skills without catastrophic forgetting due to its plasticity [100, 13, 69, 66], i.e., its ability to continually strengthen more frequently used synaptic connections and eliminate synaptic connections that are rarely used, a phenomenon called *synaptic pruning* [15]. This way, the brain generates *neural pathways* to efficiently transmit information and are used to complete different tasks [68, 60, 32, 26].

In the past, several studies have explored to mimic this behaviour by training sub-networks of a neural network for each task [91, 45, 8, 44]. This approach consists of reserving specific subsets of weights

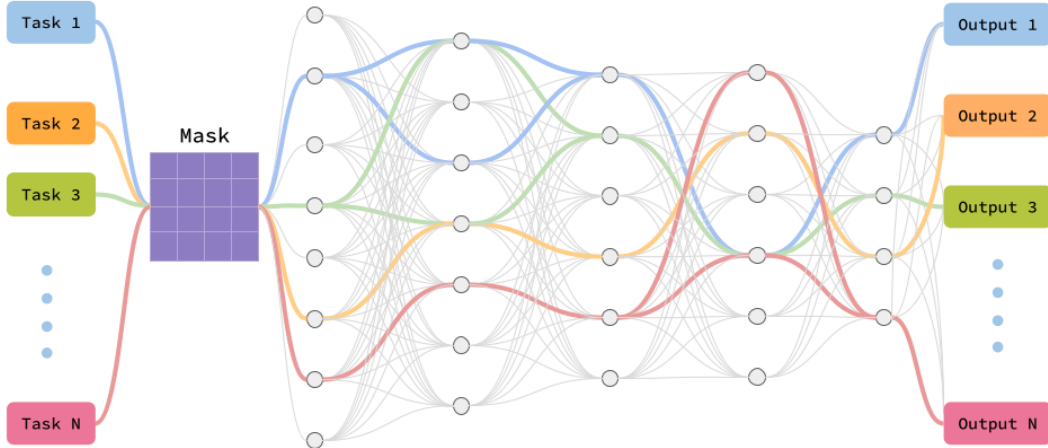

Figure 1: For any given task, our proposed method activates a specific part of the neural network.

for each task. We refer to these sub-networks as *neural pathways*. These studies operate in supervised and continual learning scenarios therefore their applicability to multi-task RL remains unexplored. Moreover, these pathways can be quite large, often comprising 30% or more of the total network parameters [44], making them inefficient. Pruning literature [18, 15, 24, 5, 56, 92, 57] has focused on identifying highly sparse sub-networks. However, these approaches are generally designed for single-task scenarios, as they specifically target and prune weights that are not utilized within the chosen sub-network. In RL, finding performant sub-networks is more challenging due to the data distribution shift during online training [27]. Our findings indicate that recent methods employing dynamic sparse training and gradient-based topology evolution to prune networks for reinforcement learning [27, 82] are ineffective at high sparsity levels (95%).

In this paper, we study the feasibility of training neural pathways for multi-task RL, where each task is tackled by a different pathway of the same underlying network. For offline RL, we show that configuring the pathway in a single shot using existing parameter importance criteria [1] is sufficient and outperforms standard multi-task training. To address distribution shifts for online RL, we introduce *data adaptive pathway discovery* (DAPD), which employs an initial warm-up phase for pathway discovery, in which the pathway is progressively reconfigured during policy training, and then kept frozen for the final stages of training. Our proposed method not only surpasses the performance of competitive dynamic sparse training baselines but also Dense networks in single and multi-task continuous control tasks.

In contrast to existing multi-task training methods in RL focusing on gradient manipulation [97, 80, 14, 7, 72], architecture modifications [93] and task similarities [96, 16, 77], we show that discovering task pathways is simple, effective and allows for energy savings. We manage to discover pathways that utilize only a small fraction (5%) of the neural network parameter. Given the high-sparsity level, the resulting policy requires low *floating point operations* (FLOPs) by potentially leveraging sparse matrix multiplication (SpMM) [54, 53, 55] to increase energy efficiency, reduce carbon footprint and potentially be deployed on low-resource devices (i.e., embedded systems, edge devices, etc.).

We highlight our contributions as follows:

- We showcase how to train multiple neural pathways for multi-task RL where the objective is to improve energy efficiency and reduce the carbon footprint associated with both offline and online RL training.

- We introduce Data Adaptive Pathway Discovery (DAPD), which leverages network sensitivity to adjust pathways in response to the data distribution shifts encountered in online RL. This capability enables us to identify pathways at high levels of sparsity and surpass competitive sparse training baselines [27, 82].

- We demonstrate superior sample efficiency and performance in both single and multi-task RL. The sparsity in the model induces 20x increase in energy efficiency compared to alternative approaches, achieved through FLOP count reduction and the utilization of Sparse Matrix Multiplication (SpMM).

## 2 Related Work

**Sparse Networks**   Advances in finding sparse networks have proven that there exists sub-networks which contain a small fraction of the parameters of the Dense deep neural network yet retain similar performance [18, 15, 24, 5, 56, 92]. Building upon a three-decade-old saliency criterion used for pruning trained models [48], a recent technique to prune models at initialization was proposed by [38] and was swiftly followed by subsequent works [90, 10, 83] which can find sub-networks at initialization and operate in supervised learning. In *offline* RL, pruned models at initialization has been proven effective [3], but are limited to a static and pre-defined training dataset. Recent studies on sparsity in RL methods suggest leveraging gradient-based topology evolution criteria, as proposed in RiGL [27], to identify sparse networks. Rlx2 [82] shows that topology evolution encounters challenges in maintaining stable value estimation in continuous control tasks. In response to this limitation, RLx2 incorporates a multi-step temporal difference (TD) target [81] and a dynamic capacity on replay buffer.  These models exhibit high sensitivity to specific sparsity levels for continuous control tasks, necessitating careful tuning for optimal performance.

**Multi-task RL**   The motivation of many works in *multi-task* RL is that training a policy for more than one task becomes difficult due to gradient interference, i.e., gradients for different tasks pointing in very different directions [97, 41].  Recent work proposes several possible solutions, such as constraining the conflicting gradient update [97, 80, 14, 7, 72], constraining data sharing among irrelevant tasks [96, 16], modularize the neural network to reuse network components across tasks [67, 93] and learning the underlying context of relevant tasks [77]. Inspired by previous works on continual learning [91, 45, 8, 44], we propose to tackle the multi-task RL scenario by assigning each task to a specific sub-network of a shared deep network. We introduce a robust algorithm designed to concurrently find the sub-networks and optimize the parameters of the shared network in the context of RL.

**Energy Efficient Deep Learning**   Recent works suggest [59, 31] carbon-emission can be reduced by using sample efficient ML architecture, optimized hardware for ML (ex: TPU, GPU) or cloud compute in location with clean energy. Many hardware startups [55] are developing AI-specific chips, some of which claim to achieve a substantial increase in FLOPS/Watt gains from simply reconfiguring hardware to do the same number of operations for less economic cost. For performance per power efficiency, the FPGA-based accelerator [49] is 11.6 times better than GPU-based one.  NVIDIA [52] already shows almost 10x speed up using Block-SpMM using V100 GPU. [39] proposes a high-performance sparse-matrix library for low-precision integers on Tensor cores and shows 2.37x performance improvement on Nvidia-A100. There has been ongoing research on how to properly estimate the carbon footprint [31, 59]. We focus on utilizing the sparse matrix multiplication (SpMM) aware hardware and software to reduce the carbon footprint.

## 3 Neural Pathway Discovery for RL

In this work, we aim to show the feasibility of tackling both online and offline multi-task RL by *pathway discovery* (PD), i.e. identifying and training different sub-networks of a same shared network for each task. The sparsity of each sub-network allows for energy efficiency gains thus reducing the carbon footprint associated with training and deployment.

### 3.1 Background

**Reinforcement Learning**   We consider learning in a Markov Decision Process (MDP) defined by the tuple $(S, A, P, R)$ with states $s \in S$, actions $a \in A$, transition dynamics $P(s'|s, a)$, and reward function $R(s, a)$. At time step $t$, the state, action, and reward are denoted as $s_t$, $a_t$, and $r_t = R(s_t, a_t)$, respectively. An episode is a trajectory $\tau = (s_0, a_0, r_0, s_1, a_1, r_1, ..., s_T, a_T, r_T)$, accumulating an episodic return $R_T = \sum_{t=0}^{T} r_t$. For continuous control tasks, we use an infinite horizon, $T = \infty$, aiming to maximize the expected discounted return $\mathbb{E}[\sum_{t=0}^{\infty} \gamma^t r_t]$. Here, $\gamma$ represents the discount factor set at 0.99. In a multitask setup, we have $N$ tasks with respective reward functions $\{R_n\}_{n=1}^{N}$ and optimal policies $\{\pi_n^*(a|s)\}_{n=1}^{N}$. No assumptions are made about transition dynamics.

**Neural Pathways** Denote by $f(x, \theta)$ a neural network with initial parameters $\theta_0 \sim \Theta_\theta$. We define a binary *mask* $m \in \{0, 1\}^{|\theta|}$ which defines whether a connection in $f$ should be masked or not. Applying the mask on the parameters of $f$ creates a *neural pathway*, i.e. a sparse sub-network $\theta \odot m$, which parameterizes the resulting function $f(x, \theta \odot m)$, where $\odot$ denotes element-wise product [91]. Given a dataset of task examples $D$, the learning problem consists of learning both the mask $m$ and parameters $\theta$. In this paper, we operate in a multi-task setting and learn a pathway for each task. Therefore, we will usually have a collection of masks $\{m_n\}$ and resulting pathways $\{\theta \odot m_n\}$, indexed by the task $n$. Note that all the pathways are defined with respect to the same base parameters $\theta$, i.e. tasks compete for capacity under the same base network $f$. This introduces optimization challenges, especially in online RL, that we address with our method in the next section.

## 3.2 Data Adaptive Pathway Discovery (DAPD)

In online RL, an agent interacts with an environment, whether simulated or real-world, to gather training samples using its behaviour policy. As the policy improves, there exists a significant shift not only in the quality but also in the distribution of the collected training samples. To address such shifts in distribution, we introduce *Data Adaptive Pathway Discovery* (DAPD), which consistently adapts the sub-network as the policy advances, accommodating changes in data distribution throughout the training process. Our algorithm to find neural pathways for online RL relies on the following crucial design aspects. We will first outline the parameter selection criteria proposed in [38], followed by how we incorporate the criteria into the development of our online RL algorithm.

**Selection Criterion** Following previous work [1], we infer the task specific mask using a criterion $\mathbf{S}$, which measures the importance of every parameter in the neural network for a given task:

$$m = \mathcal{T}_k\Big(\mathbf{S}(\theta; D)\Big), \tag{1}$$

where $D$ is the dataset containing training samples for the current task, $\mathcal{T}_k$ is defined as the "Top-$k$" operator, that sets top-$k$ parameters to 1 and the rest to 0, thus controlling the percentage of total network parameters considered active.

We use a gradient-based saliency criterion that identifies important connections by using a sensitivity measure, defined as the influence of each weight on the loss function [38, 48]. Formally, the effect of weight $\theta_q$ on the loss is:

$$\mathbf{S}(\theta_q) = \lim_{\epsilon \to 0} \left| \frac{\mathcal{L}(\theta_0) - \mathcal{L}(\theta_0 + \epsilon\delta_q)}{\epsilon} \right| = \left| \theta_q \frac{\partial\mathcal{L}}{\partial\theta_q} \right|, \tag{2}$$

where $\delta_q$ is a vector whose $q$-th element equals $\theta_q$ and all other elements are 0. This scoring metric has been used in [38] to prune models at initialization by setting the parameters with scores lower than a certain threshold to 0. Here, we use this scoring metric to not only rank important parameters at initialization, but also update them to adapt to the changing data distribution.

**Adaptive Masking** In online RL, an agent interacts with the environment and collects the corresponding training dataset $D$ for a given task. Therefore, the task dataset $D$ depends on the current policy. We leverage the most recent episodic data collected by the agent. At the $j^{th}$ training step, we calculate a score $\mathbf{S}^j$ based on the most recently gathered training samples:

$$\mathbf{S}^j(\theta_q, D^{t-L:t}) = \left| \theta_q \frac{\partial\mathcal{L}(\theta_0; D^{t-L:t})}{\partial\theta_q} \right|, \tag{3}$$

where $D^{t-L:t}$ is the most recent $\{(s, a, s', r)\}_{l=0}^{L}$ tuples added to the replay buffer and $L$ is the number of timesteps of an episode of finite length. We found that scoring parameters only on the basis on the most recent samples collected by the updated policy is crucial. This translates into the assumption that as our policy improves, the quality of recently collected samples gets better. We found that it is harmful to include prior historical data for mask inference, especially in the initial stage of online training, where the agent takes random actions to explore the environment.

To prevent abrupt mask changes and ensure training stability, we propose averaging the last $K$ scores and we can express the *K-length moving average* as $\frac{1}{K}\sum_{k=0}^{K-1} \mathbf{S}^{j-k}$. Thus, to update the mask we use the following objective:

$$m = \mathcal{T}_k\Big(\frac{1}{K}\sum_{k=0}^{K-1} \mathbf{S}^{j-k}(\theta_q, D^{t-L:t})\Big). \tag{4}$$

**Warm-up and Freeze** We found it important to have an initial *warm-up* phase during training, where we periodically adjust the mask using Eq. (4) until we attain a reasonable episodic return *threshold, TH*, which is a hyper-parameter of the algorithm often used in foundational and applied RL research studies [46, 74, 81, 36]. After the warm-up phase is completed, we *freeze* the mask for the remaining training process to allow further training of the parameters in the pathway. Because of the potential presence of *many lottery sub-networks* [18], we found that freezing the mask after a warm-up phase avoids continued oscillation among different sub-networks, which reduces variation in end performance across runs. During the warm-up phase, we perform a *periodic update* of $m$. As the mask updates, the solution space for optimizing the sub-network changes accordingly. Therefore, it is important that the mask update frequency is slower than the frequency of the parameter updates. This ensures that the optimization process remains aligned with the evolving sub-network solution space resulting from changes in the mask. Thus, while we perform per-step updates of sub-network parameters, we only update the mask at the end of each episode.

**Online Algorithm** We use Soft Actor-Critic (SAC) [29] as our online algorithm. SAC defines an actor-network $\pi(\theta)$ and a critic network $Q(\phi)$, each having its own objective function (discussed in Appendix B.1). For every task $n$, we learn two separate masks $m_n^\theta$, $m_n^\phi$, each masking their corresponding base parameters, i.e. each task $n$ has two pathways $\theta \odot m_n^\theta$, $\phi \odot m_n^\phi$. We use the warm-up and freeze strategy: the masks are updated during a warm-up phase using only the most recent transitions collected from the dataset: once we reach a threshold of performance we stop updating the pathways and keep them fixed. During this phase of warm-up training, we simultaneously update the masks and the base parameters $\theta$ and $\phi$. We use Eq. (4) to update the masks. We present the pseudo-code of this procedure in Alg. 1.

---

**Algorithm 1** Multi-task SAC with DAPD (SAC-DAPD)

---
**Init:** $\pi(\theta), Q(\phi)$
**Param:** Moving Average $K$, Warm-up Threshold $TH$
▷ Episodic Return $n^{th}$ task, $R_T^n = \sum_{t=0}^{T} r_t^n$;
▷ Replay buffers $n^{th}$ task, $D_n$;
▷ Mask for $n^{th}$ tasks, $m_n = \{m_n^\theta, m_n^\phi\}$.
▷ warm-up-phase = [True, ..., True]
#Training loop:
**for** Training steps **do**
    **while** episode not done **do**
        # Collect Data:
        **for** Each Task **do**
            ▷ $a_t \sim \pi_\theta(a_t|s_t), s_{t+1} \sim T_n(s_{t+1}|s_t, a_t)$
            ▷ $D_n \leftarrow D_n \cup \{s_t, a_t, r(s_t, a_t), s_{t+1}\}$
        **end for**
        # Update Network using SAC:
        ▷ Update $\pi_\theta$ ( $\{m_1^\theta, m_2^\theta .. m_n^\theta\}, \{D_1, D_2 .. D_n\}$)
        ▷ Update $Q_\phi$ ( $\{m_1^\phi, m_2^\phi .. m_n^\phi\}, \{D_1, D_2 .. D_n\}$)
    **end while**
    # Periodically Update Masks:
    **for** Each Task **do**
        ▷ **If** warm-up-phase[$n$]:
            Update $m_n$ using Eq (4)
        ▷ **If** $R_T^n \geq TH$ : warm-up-phase[$n$] = False
    **end for**
**end for**

---

### 3.3 Pathways for Offline RL

Much like supervised training, offline RL operates with a fixed training dataset for each task. Consequently, adaptive pathway updates are unnecessary, and we can identify the top 5% most crucial weights for each task using Eq. (1). For all experiments, including baseline comparisons, we adhere to a parallel training procedure. We initiate parallel training processes [51, 46, 89], assigning one for each task and implementing asynchronous global parameter updates in a Hogwild [51] style. Convergence of the global parameter in such a method is established in the context of RL [46, 89]. Refer to Algorithm 2 for the pseudocode outlining the integration of our offline PD algorithm.

## 4   Experiments

We demonstrate the efficacy of our proposed method in learning policy for online RL tasks using only 5% of the parameters in the continuous control domain without sacrificing performance compared to the Dense counterpart. Additionally, we conduct comparisons with various baselines, evaluating the reliability of performance across different network sparsity levels. Furthermore, we investigate the use of additional parameter space to enable multitasking through multiple pathways in a multitask benchmark, concurrently presenting a more energy-efficient alternative. Environment details and snapshots are provided in Appendix C.4 and further experimental results in Appendix D.

Table 1: Performance comparison of DAPD with various baselines at **95% sparsity** in single-task experiments using MuJoCo continuous control. We compare the average episodic return over the last 10 evaluations over 5 seeds after 1 million training steps.

| Environment | SAC-Dense | RiGL | Rlx2 | SAC-DAPD |
|---|---|---|---|---|
| HalfCheetah-v2 | $8568.1 \pm 1043.56$ | $4043.95 \pm 467.88$ | $2333.31 \pm 1241.16$ | $\mathbf{9028.02 \pm 276.31}$ |
| Walker2d-v2 | $2972.49 \pm 1691.47$ | $260.3 \pm 31.16$ | $518.45 \pm 205.16$ | $\mathbf{3846.3 \pm 459.82}$ |
| Hopper-v2 | $3228.5 \pm 301.88$ | $174.89 \pm 8.12$ | $198.29 \pm 10.39$ | $\mathbf{3359.88 \pm 46.57}$ |
| Ant-v2 | $3538.22 \pm 654.76$ | $954.2 \pm 14.4$ | $963.68 \pm 6.96$ | $\mathbf{3916.65 \pm 502.82}$ |

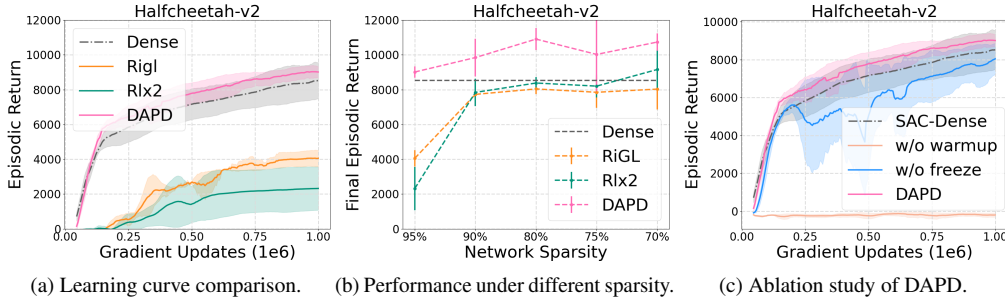

(a) Learning curve comparison.     (b) Performance under different sparsity.     (c) Ablation study of DAPD.

Figure 2: Performance comparison of DAPD with baseline on `HalfCheetah-v2`. ($a$) At **95% sparsity**, we show the learning curve of different algorithms. ($b$) Under **varying sparsity levels** we compare the average episodic return evaluated end of 1 million training steps. ($c$) Ablation study of DAPD.

## 4.1 Scenario 1: Online Single-Task RL

We conduct our experiment using *Soft-Actor-Critic* (SAC) [29] on MuJoCo continuous control tasks. We use DAPD to identify a pathway comprising only $5\%$ (or $95\%$ sparse) of the total parameters of the actor and critic network of the SAC algorithm.

**Baseline Comparison** We compare the performance of DAPD with other pruning algorithms adapted for online RL, specifically RiGL [27] and Rlx2 [82], as well as with the performance of a SAC-Dense network, in which 100% of the parameter space is trained. A comparative performance analysis involving `HalfCheetah-v2`, `Walker2d-v2`, `Hopper-v2`, and `Ant-v2` is presented in Table 1. In Figure 2 ($a$), we compare the learning curve of DAPD with SAC-Dense and with a pruning method on SAC exhibiting 95% sparsity on `HalfCheetah-v2`. We also evaluate SNIP [38] applied at initialization using Eq. (1). All the baselines are trained for the same number of steps, i.e. for DAPD the warm-up phase is included within the total training steps, with no additional training compared to the baselines. The poor performance of SNIP highlights the importance of our proposed adaptive update in an RL context.

First, we emphasize that DAPD outperforms the dense network both at the end of training, for multiple levels of sparsity, as well as during training, even with an extreme level of 95% sparsity. (Fig. 2a). A recent study [50] has highlighted the fact that dense networks exhibit primacy bias, essentially overfitting data observed early in the training. Our approach has two effects: the sparsity acts as a regularizer while training the sub-network adaptively reconfigures the parameters during the early stages of training, therefore mitigating primacy bias. RiGL and Rlx2 have specific sparsity levels for each environment at which they perform well, as shown in the Appendix (Figure 12) and in Fig 2b, but they are ineffective at a 95% sparsity level. Additional evidence regarding the learning curve and comparative analysis at various sparsity levels on `Walker2d-v2`, `Hopper-v2`, `Ant-v2` is provided in Appendix ( see Figure 13 and 14).

**Impact of Warm-Up** We hypothesize that due to the distribution shift in online training, single-shot pathway discovery techniques fail to maintain satisfactory performance. Hence, the impact of warm-up, via the periodic reconfiguration of the pathway, should be essential. To validate this hypothesis, we train SAC without warm-up: we collect random trajectories, initialize a mask using Eq (1) and train the resulting sparse SAC network for 1 million steps. We report the results in Fig. 2 ($c$), (orange), as the mean (5 seeds) performance evaluated every 5000 steps. Secondly, we show that the stopping the update of the mask during training is crucial for obtaining reliable performance.

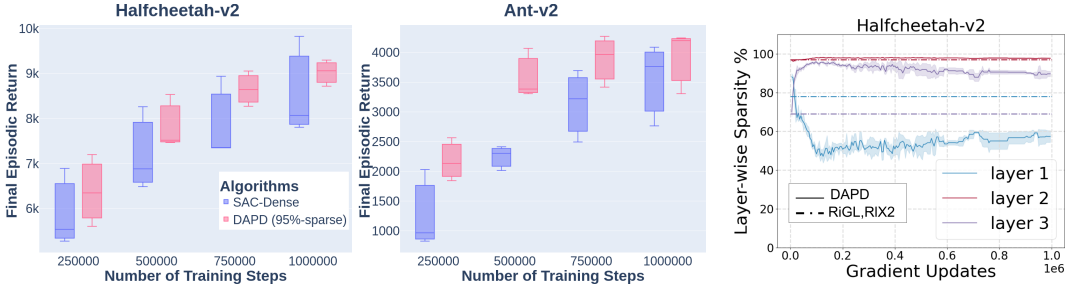

Figure 3: Sample efficiency comparison of SAC-sparse (95%) network using DAPD with the SAC-Dense counterpart. We provide a quantile plot of mean episodic return over the 10 evaluations over 5 seeds at different training steps.

Figure 4: Evolution of the layer-wise sparsity of the policy network over training steps.

Table 2: We compare the success rate of SAC-DAPD on MetaWorld 10 (MT10). We also show the *reduced* parameter complexity, FLOP and hence the energy consumption of SAC-DAPD compared to other baseline methods.

| Experiments | SAC-DAPD | SAC-Dense | PCGrad | SM | SAC+ME | CARE |
|---|---|---|---|---|---|---|
| MT10 tasks | $77 \pm 1.3$ | $49.0 \pm 7.3$ | $72.0 \pm 2.2$ | $73 \pm 4.3$ | $74 \pm 4.3$ | $\mathbf{84 \pm 5.1}$ |
| Parameter Counts | **17k** | 340k | 340k | 135k | 344k | 486k |
| FLOPs | **16.9k** | 339K | 339K | 78K | 363K | 368K |
| Energy Consumption, *Jules* | $k$ | $20k$ | $20k$ | $20k$ | $21.02k$ | $21.25k$ |

*DAPD w/o freeze* results in frequent switching among different sub-networks throughout the training process. This is evident in Figure 2, where *DAPD w/o freeze* (blue) exhibits high variance in the performance. In contrast, DAPD (pink) with a fixed warm-up phase (halted at episodic return, $TH$=8000), demonstrates low variance in episodic return and surpasses the performance of the SAC-Dense model. In Appendix D.1, we provide empirical evidence demonstrating the existence of numerous lottery subnetworks and validate that freezing any of these subnetworks can yield equally good performance. Further insights are provided in Appendix (Table 5), where we explore the impact of the warm-up period's duration and the rolling moving average on the scoring function in DAPD performance.

**Evolution of Sparsity**  We observe the evolution of layer-wise sparsity in Figure 4. Following evolution topology-based pruning, both RiGL and RlX2 initiate with a random mask. While they iteratively prune and grow connections within each layer, they maintain a fixed sparsity percentage per layer. This constraint leads to higher sparsity in the input layer compared to the output layer (Layer-1 and 3 in Figure 4, potentially overlooking critical features at the input layer. In contrast, DAPD examines the importance of weights across the entire network, granting us the flexibility to focus on weights throughout the network, providing a higher degree of freedom. While our initial observation reveals a sparser input layer compared to the output layer, this balance shifts as training progresses. In this specific experiment, we continuously re-calibrate the mask during training. Our findings indicate that while the sparsity adapts, it fluctuates within a certain range, reinforcing our rationale for maintaining a fixed-length warm-up phase.

**Sample Efficiency**  To investigate the sample efficiency of DAPD, we conducted experiments with varying numbers of training steps, comparing the performance with the SAC-Dense model in MuJoCo tasks. In Figure 3 we find DAPD to consistently surpass the performance of the SAC-Dense model, proving DAPD to be more sample efficient. This highlights its potential for more efficient resource utilization compared to its Dense counterpart. Additional experimental results are provided in Figure 15 in Appendix.

## 4.2  Scenario 2: Online Multi-Task RL

We compare the performance of our proposed method against various baselines on the MetaWorld [98] MT10 benchmark. The agent gets a binary score based on success in reaching an expected goal state. In Table 2, we evaluate the mean success rate over 10 tasks. Results include mean and standard deviation over 10 seeds.

**Baseline Comparison** We compare our method to vanilla SAC (SAC-Dense), SM [93] which requires a modified network architectures to route through modular networks, PCGrad [97], which requires a complex gradient update procedure, and CARE [77], which uses pre-trained language model to encode meta-data and retrieve contextual information about the tasks. Despite its simplicity, we observe a significant performance enhancement with DAPD compared to Dense, from $49\%$ to a remarkable success rate of $77\%$. This improvement is particularly impressive considering that DAPD uses the same learning objective and gradient update rules as the Dense model while operating with an extremely sparse network, utilizing only 5% of the model weights for any task.

**Energy Consumption** We estimate energy consumption based on FLOP counts [23] and create a normalized (on a scale of 1) energy consumption profile alongside performance in Figure 5. A reduction in FLOP counts, leads to decreased computations and lower energy costs, resulting in a proportional relationship between FLOP count and energy cost in *Joules* i.e. FLOPs $\propto$ Joules. This allows us to illustrate the trade-off in energy consumption during inference on Dense networks for marginal gain. Compared to other baseline methods, our proposed DAPD can potentially save 20x energy usage while achieving competitive performance. We would like to point out that the only baseline that exceeds our performance is CARE [77] which requires a pre-trained language model and assumes task dependency.

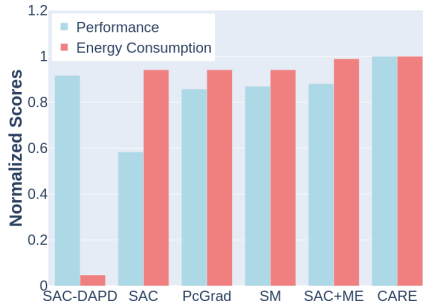

Figure 5: Energy consumption profile of algorithms on MetaWorld benchmark. We normalize the performance and energy consumption to highlight the trade-off for performance gain.

We discover that a warm-up phase of $10k$ steps, constituting $0.5\%$ of the total training steps, proves effective in configuring the pathway. This approach avoids gradient interference encountered due to multitask training and prevents catastrophic failures in the learning process. Further details are provided in Appendix D.2. We consider choosing the neural pathways for each task independently from one another and yet we find considerable pathway overlap. Further discussed in Appendix D.4. Since this is the first work that proposes the neural pathways in a multitask RL setting, there is a range of open questions that need to be explored. Limitations of our method and possible avenues of future work are discussed in the Appendix E.

### 4.3 Scenario 3: Offline Multi-Task RL

For our offline experiments, we train SAC to collect the training dataset ( see Appendix C.2). We use BCQ [22] and IQL [37] for our offline experiments. We compare the performance of PD with two Dense baseline methods: $(a)$ a multi-task variant (MT) that uses a single Dense network for multiple tasks, and $(b)$ a multi-head multi-task variant (MHMT) that employs independent heads for each task, both utilizing the Dense network. It is crucial to emphasize that, similar to MT and MHMT, we refrain from imposing any additional multitask learning objectives, facilitating a fair baseline comparison. Instead, our approach provides the network with the advantage of training separate parameters for each task. We evaluate performance and energy efficiency in the MetaWorld benchmark. We further examine robustness in comparison to Dense baselines under mixed data distribution and sample complexity on `HalfCheetah`-Multitask. Supplementary experiments, including comparisons with single-task experts on additional sets of tasks, are presented in Appendix D.3.

**MetaWorld Benchmark** The success rate (mean and standard deviation over 10 seeds) is reported in Table 3. Whereas baselines with Dense networks suffer from high variance in performance, with BCQ+PD, we get a perfect score on all MetaWorld tasks. A success rate of 100 indicates BCQ reaches the goal for all the tasks all the time. Moreover, if our model uses $\mathcal{K}$ *Joules* of energy, based on the FLOP relationship, we show how costly other baselines are in terms of energy consumption. We also provide a detailed breakdown of individual task performance Appendix (Table 9), where we compare *Conservative data sharing* (CDS) [96], an offline-multitask learning algorithm that utilizes task similarity.

Table 3: Performance comparison in MetaWorld offline. We compare the final success rate (mean and std over 10 seeds) of pathways discovery (PD) on MT10 tasks with Offline-MT and Offline-MHMT baselines on offline RL algorithms. We also show the *reduced parameter complexity* (x times) of PD compared to other baseline method

| Experiment | Offline PD | | Offline MT | | Offline MHMT | |
|---|---|---|---|---|---|---|
| | BCQ | IQL | BCQ | IQL | BCQ | IQL |
| MT-10 tasks | **100 ± 0.0** | **97.3 ± 7.17** | $81.5 \pm 24.15$ | $79.1 \pm 26.81$ | $95.9 \pm 10.44$ | $96.5 \pm 7.10$ |
| Parameter Counts | **67k** | **54k** | 1.34M | 1.01M | 1.38M | 1.12M |
| FLOPs | **29.4K** | **53.6k** | 589K | 1073K | 629k | 1128k |
| Energy Consumption, *Joules* | $k$ | $k$ | $20k$ | $20k$ | $21.25k$ | $21.02k$ |

**Performance Under Mixed Data Distribution**   To further validate the reliability of our method, we conduct a *sample complexity analysis* [2] across varying reduced training sample sizes. The interquartile plot of normalized scores in Figure 6(a) consistently demonstrates an improvement when BCQ is trained with PD.

**Performance Under Sample Complexity**   Even within the Offline setting, it is anticipated that there may be a distribution shift in the static training dataset. Therefore, the resilience of offline algorithms is assessed under a mixed data distribution [22, 21, 28]. In Figure 6(b), we demonstrate that PD exhibits enhanced performance even when confronted with a mixed dataset. Additional details regarding data collection and the experimental setup are provided in the Appendix D.3.2.

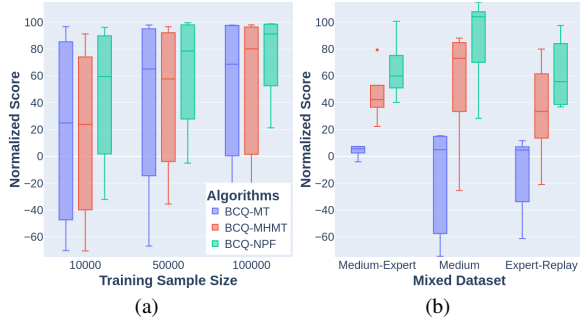

Figure 6: (a) We compare the sample complexity analysis of BCQ+PD with BCQ-MT (Multitask) and BCQ-MHMT (Multi-head Multitask) baselines on `HalfCheetah` multitask. (b) Performance (Normalized Score) plot of BCQ on `HalfCheetah` multitask trained with PD with baselines on mixed data distribution.

## 5   Conclusion

We propose to train task-specific neural pathways for reinforcement learning (RL) within a single deep neural network. We presented the *Data Adaptive Pathway Discovery* (DAPD), which builds on the pruning literature [18, 38, 90, 15] to identify individual pathways. The DAPD method overcomes limitations in sparse network discovery using connection sensitivity under dynamic data distributions. Our methodology showcases superior sample efficiency and excels in both single and multitask training instances over dense networks, while using $95\%$ fewer parameters on high-dimensional continuous controls, without the need for complex objective functions or gradient manipulation. Beyond performance gains, our work has the potential to significantly enhance energy efficiency in neural networks. Through reduced FLOP counts and leveraging Sparse Matrix Multiplication (SpMM), our approach can be $20x$ more energy efficient than alternative strategies.

This work opens up research into the possibility of a single neural network being trained for multiple purposes in RL (i.e. learning different features, multiple value functions etc.) while providing better efficiency in utilizing the parameter space of neural networks. While our method can be integrated into other selective data-sharing and gradient update methods, further research is required in this direction. Expanding the scope of experiments would provide valuable insights for future investigations.

# 6 Acknowledgement

We want to thank Raihan Siraj, Rishabh Agarwal for their helpful discussion and suggestions and the anonymous reviewers for their helpful comments. This work is supported by the Natural Sciences and Engineering Research Council of Canada (NSERC) Grant, Canadian Institute for Advanced Research (CIFAR) Grant. Samin Yeasar Arnob would like to thank the generous support of the DeepMind Graduate Award and Unifying Neuroscience and Artificial Intelligence in Quebec (UNIQUE).

Riyasat Ohib and Sergey Plis were in part supported by NIH R01DA040487 and NSF 2112455.

This research was enabled by the compute support provided by Calcul Quebec, the Digital Research Alliance of Canada and Mila Quebec AI Institute.

# References

[1] Milad Alizadeh. "Single-shot network pruning (SNIP) in PyTorch". In: *GitHub* (2019). URL: https://github.com/mil-ad/snip.

[2] Samin Yeasar Arnob, Riashat Islam, and Doina Precup. "Importance of Empirical Sample Complexity Analysis for Offline Reinforcement Learning". In: *CoRR* abs/2112.15578 (2021). arXiv: 2112.15578. URL: https://arxiv.org/abs/2112.15578.

[3] Samin Yeasar Arnob et al. "Single-Shot Pruning for Offline Reinforcement Learning". In: *arXiv preprint arXiv:2112.15579* (2021).

[4] Christopher Berner et al. "Dota 2 with Large Scale Deep Reinforcement Learning". In: *CoRR* abs/1912.06680 (2019). arXiv: 1912.06680. URL: http://arxiv.org/abs/1912.06680.

[5] Davis Blalock et al. "What is the state of neural network pruning?" In: *Proceedings of machine learning and systems* 2 (2020), pp. 129–146.

[6] Cerebras. "Accelerating Large GPT Training with Sparse Pre-Training and Dense Fine-Tuning". In: (2023).

[7] Zhao Chen et al. "GradNorm: Gradient Normalization for Adaptive Loss Balancing in Deep Multitask Networks". In: *CoRR* abs/1711.02257 (2017). arXiv: 1711.02257. URL: http://arxiv.org/abs/1711.02257.

[8] Brian Cheung et al. "Superposition of many models into one". In: *CoRR* abs/1902.05522 (2019). arXiv: 1902.05522. URL: http://arxiv.org/abs/1902.05522.

[9] Aakanksha Chowdhery et al. *PaLM: Scaling Language Modeling with Pathways*. 2022. arXiv: 2204.02311 [cs.CL].

[10] Pau De Jorge et al. "Progressive skeletonization: Trimming more fat from a network at initialization". In: *arXiv preprint arXiv:2006.09081* (2020).

[11] Lei Deng et al. "Model Compression and Hardware Acceleration for Neural Networks: A Comprehensive Survey". In: *Proceedings of the IEEE* 108.4 (2020), pp. 485–532. DOI: 10.1109/JPROC.2020.2976475.

[12] Jacob Devlin et al. "BERT: Pre-training of Deep Bidirectional Transformers for Language Understanding". In: *CoRR* abs/1810.04805 (2018). arXiv: 1810.04805. URL: http://arxiv.org/abs/1810.04805.

[13] Daniel Drubach. *The brain explained*. Pearson, 2000.

[14] Yunshu Du et al. *Adapting Auxiliary Losses Using Gradient Similarity*. 2020. arXiv: 1812.02224 [stat.ML].

[15] Irwin Feinberg. "Schizophrenia: caused by a fault in programmed synaptic elimination during adolescence?" In: *Journal of psychiatric research* 17.4 (1982), pp. 319–334.

[16] Christopher Fifty et al. *Efficiently Identifying Task Groupings for Multi-Task Learning*. 2021. arXiv: 2109.04617 [cs.LG].

[17] Chelsea Finn, Pieter Abbeel, and Sergey Levine. "Model-Agnostic Meta-Learning for Fast Adaptation of Deep Networks". In: *CoRR* abs/1703.03400 (2017). arXiv: 1703.03400. URL: http://arxiv.org/abs/1703.03400.

[18] Jonathan Frankle and Michael Carbin. *The Lottery Ticket Hypothesis: Finding Sparse, Trainable Neural Networks*. 2019. arXiv: 1803.03635 [cs.LG].

[19] Jonathan Frankle et al. "Linear mode connectivity and the lottery ticket hypothesis". In: *International Conference on Machine Learning*. PMLR. 2020, pp. 3259–3269.

[20] Charlotte Freitag et al. "The real climate and transformative impact of ICT: A critique of estimates, trends, and regulations". In: *Patterns* 2.9 (2021), p. 100340. ISSN: 2666-3899. DOI: https://doi.org/10.1016/j.patter.2021.100340. URL: https://www.sciencedirect.com/science/article/pii/S2666389921001884.

[21] Justin Fu et al. "D4RL: Datasets for Deep Data-Driven Reinforcement Learning". In: *CoRR* abs/2004.07219 (2020). arXiv: 2004.07219. URL: https://arxiv.org/abs/2004.07219.

[22] Scott Fujimoto, David Meger, and Doina Precup. "Off-Policy Deep Reinforcement Learning without Exploration". In: *CoRR* abs/1812.02900 (2018). arXiv: 1812.02900. URL: http://arxiv.org/abs/1812.02900.

[23] FAIR fvcore. "fvcore". In: *GitHub* (2022). URL: https://github.com/facebookresearch/fvcore.

[24] Trevor Gale, Erich Elsen, and Sara Hooker. "The state of sparsity in deep neural networks". In: *arXiv preprint arXiv:1902.09574* (2019).

[25] Trevor Gale et al. "Sparse GPU Kernels for Deep Learning". In: *CoRR* abs/2006.10901 (2020). arXiv: 2006.10901. URL: https://arxiv.org/abs/2006.10901.

[26] Melvyn A. Goodale et al. "Separate neural pathways for the visual analysis of object shape in perception and prehension". In: *Current Biology* 4.7 (1994), pp. 604–610. ISSN: 0960-9822. DOI: https://doi.org/10.1016/S0960-9822(00)00132-9. URL: https://www.sciencedirect.com/science/article/pii/S0960982200001329.

[27] Laura Graesser et al. *The State of Sparse Training in Deep Reinforcement Learning*. 2022. arXiv: 2206.10369 [cs.LG].

[28] Çaglar Gülçehre et al. "RL Unplugged: Benchmarks for Offline Reinforcement Learning". In: *CoRR* abs/2006.13888 (2020). arXiv: 2006.13888. URL: https://arxiv.org/abs/2006.13888.

[29] Tuomas Haarnoja et al. "Soft Actor-Critic: Off-Policy Maximum Entropy Deep Reinforcement Learning with a Stochastic Actor". In: *CoRR* abs/1801.01290 (2018). arXiv: 1801.01290. URL: http://arxiv.org/abs/1801.01290.

[30] Yihui He and Song Han. "ADC: Automated Deep Compression and Acceleration with Reinforcement Learning". In: *CoRR* abs/1802.03494 (2018). arXiv: 1802.03494. URL: http://arxiv.org/abs/1802.03494.

[31] Peter Henderson et al. "Towards the Systematic Reporting of the Energy and Carbon Footprints of Machine Learning". In: *CoRR* abs/2002.05651 (2020). arXiv: 2002.05651. URL: https://arxiv.org/abs/2002.05651.

[32] Uwe Jürgens. "Neural pathways underlying vocal control". In: *Neuroscience & Biobehavioral Reviews* 26.2 (2002), pp. 235–258.

[33] Jared Kaplan et al. "Scaling Laws for Neural Language Models". In: *CoRR* abs/2001.08361 (2020). arXiv: 2001.08361. URL: https://arxiv.org/abs/2001.08361.

[34] Diederik P Kingma and Max Welling. *Auto-Encoding Variational Bayes*. 2014. arXiv: 1312.6114 [stat.ML].

[35] Diederik P Kingma and Max Welling. *Auto-Encoding Variational Bayes*. 2014. arXiv: 1312.6114 [stat.ML].

[36] Jens Kober, J Andrew Bagnell, and Jan Peters. "Reinforcement learning in robotics: A survey". In: *The International Journal of Robotics Research* 32.11 (2013), pp. 1238–1274.

[37] Ilya Kostrikov, Ashvin Nair, and Sergey Levine. "Offline Reinforcement Learning with Implicit Q-Learning". In: *CoRR* abs/2110.06169 (2021). arXiv: 2110.06169. URL: https://arxiv.org/abs/2110.06169.

[38] Namhoon Lee, Thalaiyasingam Ajanthan, and Philip HS Torr. "Snip: Single-shot network pruning based on connection sensitivity". In: *arXiv preprint arXiv:1810.02340* (2018).

[39] Shigang Li, Kazuki Osawa, and Torsten Hoefler. "Efficient Quantized Sparse Matrix Operations on Tensor Cores". In: *SC22: International Conference for High Performance Computing, Networking, Storage and Analysis*. IEEE, Nov. 2022. DOI: 10.1109/sc41404.2022.00042. URL: https://doi.org/10.1109%2Fsc41404.2022.00042.

[40] Tianyu Li et al. "Model-based Motion Imitation for Agile, Diverse and Generalizable Quadupedal Locomotion". In: *CoRR* abs/2109.13362 (2021). arXiv: 2109.13362. URL: https://arxiv.org/abs/2109.13362.

[41] Bo Liu et al. "Conflict-Averse Gradient Descent for Multi-task Learning". In: *CoRR* abs/2110.14048 (2021). arXiv: 2110.14048. URL: https://arxiv.org/abs/2110.14048.

[42] Lin Qiao Liu. "Building a Sparse Convolutional Neural Network Inference Accelerator on Field-Programmable Gate Arrays". PhD thesis. University of Toronto (Canada), 2021.

[43] Yinhan Liu et al. "RoBERTa: A Robustly Optimized BERT Pretraining Approach". In: *CoRR* abs/1907.11692 (2019). arXiv: 1907.11692. URL: http://arxiv.org/abs/1907.11692.

[44] Arun Mallya and Svetlana Lazebnik. "PackNet: Adding Multiple Tasks to a Single Network by Iterative Pruning". In: *CoRR* abs/1711.05769 (2017). arXiv: 1711.05769. URL: http://arxiv.org/abs/1711.05769.

[45] Arun Mallya and Svetlana Lazebnik. "Piggyback: Adding Multiple Tasks to a Single, Fixed Network by Learning to Mask". In: *CoRR* abs/1801.06519 (2018). arXiv: 1801.06519. URL: http://arxiv.org/abs/1801.06519.

[46] Volodymyr Mnih et al. "Asynchronous methods for deep reinforcement learning". In: *International conference on machine learning*. PMLR. 2016, pp. 1928–1937.

[47] Volodymyr Mnih et al. "Playing Atari with Deep Reinforcement Learning". In: *CoRR* abs/1312.5602 (2013). arXiv: 1312.5602. URL: http://arxiv.org/abs/1312.5602.

[48] Michael C Mozer and Paul Smolensky. "Skeletonization: A technique for trimming the fat from a network via relevance assessment". In: *Advances in neural information processing systems* 1 (1988).

[49] Hiroki Nakahara et al. "FPGA-Based Training Accelerator Utilizing Sparseness of Convolutional Neural Network". In: *2019 29th International Conference on Field Programmable Logic and Applications (FPL)*. 2019, pp. 180–186. DOI: 10.1109/FPL.2019.00036.

[50] Evgenii Nikishin et al. "The primacy bias in deep reinforcement learning". In: *International conference on machine learning*. PMLR. 2022, pp. 16828–16847.

[51] Feng Niu et al. *HOGWILD!: A Lock-Free Approach to Parallelizing Stochastic Gradient Descent*. 2011. arXiv: 1106.5730 [math.OC].

[52] Nvidia. "Accelerating Matrix Multiplication with Block Sparse Format and NVIDIA Tensor Cores". In: (2021). URL: https://developer.nvidia.com/blog/accelerating-matrix-multiplication-with-block-sparse-format-and-nvidia-tensor-cores/.

[53] Nvidia. "CUTLASS: Fast Linear Algebra in CUDA C++". In: (2017). URL: https://developer.nvidia.com/blog/cutlass-linear-algebra-cuda/.

[54] Nvidia. "How Sparsity Adds Umph to AI Inference". In: (2020). URL: https://blogs.nvidia.com/blog/2020/05/14/sparsity-ai-inference/.

[55] NYTimes. "Big Bets on A.I. Open a New Frontier for Chip Start-Ups, Too". In: (2018). URL: https://www.nytimes.com/2018/01/14/technology/artificial-intelligence-chip-start-ups.html.

[56] Riyasat Ohib et al. "Explicit Group Sparse Projection with Applications to Deep Learning and NMF". In: *Transactions on Machine Learning Research* (2022). URL: https://openreview.net/forum?id=jIrOeWjdpc.

[57] Riyasat Ohib et al. "Unmasking Efficiency: Learning Salient Sparse Models in Non-IID Federated Learning". In: *arXiv preprint arXiv:2405.09037* (2024).

[58] Adam Paszke et al. "PyTorch: An Imperative Style, High-Performance Deep Learning Library". In: *Advances in Neural Information Processing Systems 32*. Ed. by H. Wallach et al. Curran Associates, Inc., 2019, pp. 8024–8035.

[59] David Patterson et al. "The Carbon Footprint of Machine Learning Training Will Plateau, Then Shrink". In: *Computer* 55.7 (2022), pp. 18–28. DOI: 10.1109/MC.2022.3148714.

[60] Tomáš Paus et al. "Structural maturation of neural pathways in children and adolescents: in vivo study". In: *Science* 283.5409 (1999), pp. 1908–1911.

[61] Xue Bin Peng et al. "Learning Agile Robotic Locomotion Skills by Imitating Animals". In: *CoRR* abs/2004.00784 (2020). arXiv: 2004.00784. URL: https://arxiv.org/abs/2004.00784.

[62] PyTorch. "Fast Block Sparse Matrices for Pytorch". In: (2017). URL: https://github.com/huggingface/pytorch_block_sparse.

[63] Kate Rakelly et al. "Efficient Off-Policy Meta-Reinforcement Learning via Probabilistic Context Variables". In: *CoRR* abs/1903.08254 (2019). arXiv: 1903.08254. URL: http://arxiv.org/abs/1903.08254.

[64] Aditya Ramesh et al. "Zero-Shot Text-to-Image Generation". In: *CoRR* abs/2102.12092 (2021). arXiv: 2102.12092. URL: https://arxiv.org/abs/2102.12092.

[65] Alex Renda, Jonathan Frankle, and Michael Carbin. "Comparing Rewinding and Fine-tuning in Neural Network Pruning". In: *CoRR* abs/2003.02389 (2020). arXiv: 2003.02389. URL: https://arxiv.org/abs/2003.02389.

[66] Lucy B. Rorke. "Central Nervous System Plasticity and Repair". In: *Journal of Neuropathology & Experimental Neurology* 44.5 (Sept. 1985), pp. 530–530. ISSN: 0022-3069. DOI: 10.1097/00005072-198509000-00008. eprint: https://academic.oup.com/jnen/article-pdf/44/5/530/8142078/44-5-530.pdf. URL: https://doi.org/10.1097/00005072-198509000-00008.

[67] Clemens Rosenbaum, Tim Klinger, and Matthew Riemer. "Routing Networks: Adaptive Selection of Non-linear Functions for Multi-Task Learning". In: *CoRR* abs/1711.01239 (2017). arXiv: 1711.01239. URL: http://arxiv.org/abs/1711.01239.

[68] Peter H Rudebeck et al. "Separate neural pathways process different decision costs". In: *Nature neuroscience* 9.9 (2006), pp. 1161–1168.

[69] Jill Sakai. "Core Concept: How synaptic pruning shapes neural wiring during development and, possibly, in disease". In: *Proceedings of the National Academy of Sciences* 117.28 (2020), pp. 16096–16099. ISSN: 0027-8424. DOI: 10.1073/pnas.2010281117. eprint: https://www.pnas.org/content/117/28/16096.full.pdf. URL: https://www.pnas.org/content/117/28/16096.

[70] John Schulman et al. "Proximal Policy Optimization Algorithms". In: *CoRR* abs/1707.06347 (2017). arXiv: 1707.06347. URL: http://arxiv.org/abs/1707.06347.

[71] Roy Schwartz et al. "Green AI". In: *CoRR* abs/1907.10597 (2019). arXiv: 1907.10597. URL: http://arxiv.org/abs/1907.10597.

[72] Ozan Sener and Vladlen Koltun. "Multi-Task Learning as Multi-Objective Optimization". In: *CoRR* abs/1810.04650 (2018). arXiv: 1810.04650. URL: http://arxiv.org/abs/1810.04650.

[73] David Silver et al. "Mastering Chess and Shogi by Self-Play with a General Reinforcement Learning Algorithm". In: *CoRR* abs/1712.01815 (2017). arXiv: 1712.01815. URL: http://arxiv.org/abs/1712.01815.

[74] David Silver et al. "Mastering the game of go without human knowledge". In: *nature* 550.7676 (2017), pp. 354–359.

[75] Laura Smith et al. "Legged Robots that Keep on Learning: Fine-Tuning Locomotion Policies in the Real World". In: *CoRR* abs/2110.05457 (2021). arXiv: 2110.05457. URL: https://arxiv.org/abs/2110.05457.

[76] Shaden Smith et al. "Using DeepSpeed and Megatron to Train Megatron-Turing NLG 530B, A Large-Scale Generative Language Model". In: *CoRR* abs/2201.11990 (2022). arXiv: 2201.11990. URL: https://arxiv.org/abs/2201.11990.

[77] Shagun Sodhani et al. "Multi-Task Reinforcement Learning with Context-based Representations". In: *CoRR* abs/2102.06177 (2021). arXiv: 2102.06177. URL: https://arxiv.org/abs/2102.06177.

[78] Emma Strubell, Ananya Ganesh, and Andrew McCallum. "Energy and Policy Considerations for Deep Learning in NLP". In: *CoRR* abs/1906.02243 (2019). arXiv: 1906.02243. URL: http://arxiv.org/abs/1906.02243.

[79] Emma Strubell, Ananya Ganesh, and Andrew McCallum. "Energy and Policy Considerations for Deep Learning in NLP". In: *CoRR* abs/1906.02243 (2019). arXiv: 1906.02243. URL: http://arxiv.org/abs/1906.02243.

[80] Mihai Suteu and Yike Guo. "Regularizing Deep Multi-Task Networks using Orthogonal Gradients". In: *CoRR* abs/1912.06844 (2019). arXiv: 1912.06844. URL: http://arxiv.org/abs/1912.06844.

[81] Richard S Sutton and Andrew G Barto. *Reinforcement learning: An introduction*. MIT press, 2018.

[82] Yiqin Tan et al. *RLx2: Training a Sparse Deep Reinforcement Learning Model from Scratch*. 2023. arXiv: 2205.15043 [cs.LG].

[83] Hidenori Tanaka et al. "Pruning neural networks without any data by iteratively conserving synaptic flow". In: *Advances in neural information processing systems* 33 (2020), pp. 6377–6389.

[84] Hidenori Tanaka et al. "Pruning neural networks without any data by iteratively conserving synaptic flow". In: *CoRR* abs/2006.05467 (2020). arXiv: 2006.05467. URL: https://arxiv.org/abs/2006.05467.

[85] Garrett Thomas. "HalfCheetah Goal Velocity and Multitask". In: *GitHub* (2020). URL: https : / / github . com / gwthomas / force / commit / 9d1771320e825b1c0deced46b73b5ce0536d409b.

[86] Garrett Thomas. "Implicit Q-Learning (IQL) in PyTorch". In: *GitHub* (2021). URL: https://github.com/gwthomas/IQL-PyTorch.

[87] Neil C. Thompson et al. "Deep Learning's Diminishing Returns: The Cost of Improvement is Becoming Unsustainable". In: *IEEE Spectrum* 58.10 (2021), pp. 50–55. DOI: 10.1109/MSPEC.2021.9563954.

[88] Neil C. Thompson et al. "The Computational Limits of Deep Learning". In: *CoRR* abs/2007.05558 (2020). arXiv: 2007.05558. URL: https://arxiv.org/abs/2007.05558.

[89] John N Tsitsiklis. "Asynchronous stochastic approximation and Q-learning". In: *Machine learning* 16.3 (1994), pp. 185–202.

[90] Chaoqi Wang, Guodong Zhang, and Roger Grosse. "Picking winning tickets before training by preserving gradient flow". In: *arXiv preprint arXiv:2002.07376* (2020).

[91] Mitchell Wortsman et al. "Supermasks in superposition". In: *Advances in Neural Information Processing Systems* 33 (2020), pp. 15173–15184.

[92] Huanrui Yang, Wei Wen, and Hai Li. "Deephoyer: Learning sparser neural network with differentiable scale-invariant sparsity measures". In: *arXiv preprint arXiv:1908.09979* (2019).

[93] Ruihan Yang et al. "Multi-Task Reinforcement Learning with Soft Modularization". In: *CoRR* abs/2003.13661 (2020). arXiv: 2003.13661. URL: https://arxiv.org/abs/2003.13661.

[94] Yuxiang Yang et al. "Fast and Efficient Locomotion via Learned Gait Transitions". In: *CoRR* abs/2104.04644 (2021). arXiv: 2104.04644. URL: https://arxiv.org/abs/2104.04644.

[95] Denis Yarats and Ilya Kostrikov. *Soft Actor-Critic (SAC) implementation in PyTorch*. https://github.com/denisyarats/pytorch_sac. 2020.

[96] Tianhe Yu et al. "Conservative Data Sharing for Multi-Task Offline Reinforcement Learning". In: *CoRR* abs/2109.08128 (2021). arXiv: 2109.08128. URL: https://arxiv.org/abs/2109.08128.

[97] Tianhe Yu et al. "Gradient Surgery for Multi-Task Learning". In: *CoRR* abs/2001.06782 (2020). arXiv: 2001.06782. URL: https://arxiv.org/abs/2001.06782.

[98] Tianhe Yu et al. "Meta-World: A Benchmark and Evaluation for Multi-Task and Meta Reinforcement Learning". In: *CoRR* abs/1910.10897 (2019). arXiv: 1910.10897. URL: http://arxiv.org/abs/1910.10897.

[99] Chaoyang Zhu et al. *An Efficient Hardware Accelerator for Structured Sparse Convolutional Neural Networks on FPGAs*. 2020. arXiv: 2001.01955 [eess.SY].

[100] Karl Zilles. "Neuronal plasticity as an adaptive property of the central nervous system". In: *Annals of Anatomy-Anatomischer Anzeiger* 174.5 (1992), pp. 383–391.

# Appendix

## A   Additional Related Works

For MetaWorld online experiments, we compare the algorithms reported in the benchmark [77]. [77] focuses on the importance of contextual information and proposes *Contextual Attention-based Representation* (CARE), using a pre-trained language model [43] to encode meta-data and retrieve contextual information about the task. The contextual information added with the environment observation is used to train *SAC with Mixture of encoders* (SAC+ME). As seen in Table 2, the success of CARE largely depends on metadata which is not always available in all multitask settings. *Soft-modularization* (SM) proposes effective sharing and reusing network components across tasks. As reported in [77], the performance of SM [93] varies largely when evaluated over 10 different sub-tasks. We also include *Projecting Conflicting Gradient* (PCGrad, [97]), which focuses on tackling gradient interference. If gradients for different tasks point away from one another, PCGrad alters the gradient direction to mitigate this interference.

There is a range of techniques to find such *sparse-network* through an *iterative update* during training [18, 15, 24, 5, 56, 92]. Building upon a three-decade-old saliency criterion used for pruning trained models [48], a recent technique to prune models at initialization was proposed by [38] and was swiftly followed by newer works [90, 10, 83] which can find sub-networks at initialization. There are recent works in both offline [3] and online RL [27, 82] that discovers sparse network for RL agent.

## B   Additional Theoretical Background

### B.1   Offline and Online Theory

[22] highlight the fact that, since value estimation is trained with a fixed dataset, it provides an erroneous estimation when the policy takes an action which is out of distribution from the dataset on which the value function is trained. To overcome extrapolation error, BCQ proposes batch-constrained learning, where agents are trained to maximize reward while minimizing the mismatch between the state-action visitation of the policy and the state-action pairs contained in the batch. For a given state, BCQ uses a generative model $G_w$, e.g. a Variational Auto-encoder [34], to generate $n$ actions with high similarity to the batch dataset, and then it selects the action for which it gets the highest value: $\pi(s) = \arg\max_{A_i} Q_\phi(s, a_i)$, where $A_i \sim \{G_w(s)\}_{i=1}^n$. $G_w$ is trained to minimize the KL divergence with the actions sampled from batch dataset [35]. The action-value function or $Q$-function, $Q_\phi$ learned through minimizing TD-error is $J(Q_\phi) = \mathbb{E}_{\{s,a,s'\} \sim D, a' \sim \pi_\theta}[(r(s,a) + \gamma \hat{Q}_\phi(s',a')) - Q_\phi(s,a)^2]$. In our experiments, we also consider a variation of BCQ (denoted as BCQ-v2) in which we sample actions only using a VAE ($\pi(s) = G_w(s)$).

Since Offline RL faces a distribution shift in its value estimation due to the different distribution of the policy and expert sample, *Implicit Q-learning* (IQL) [37] proposes to avoid estimating the value for policy distribution. Instead, it trains the action-value function $Q_\phi$ using a SARSA-style update, thus enabling multi-step dynamic programming updates. IQL uses expectile regression to predict an upper expectile of TD target that approximates the maximum of $r(s,a) + \gamma[\hat{Q}_\phi(s',a')]$. IQL uses a separate value function by fitting upper expectile $V_\psi$ using the objective function: $J(V_\psi) = \mathbb{E}_{s,a \sim D}[\mathcal{L}_2^\tau(\hat{Q}_\phi(s',a') - V_\psi(s))]$, where $\mathcal{L}_2^\tau$ is asymmetric least squares. This value is used to update $Q$ function using: $J(Q_\phi) = \mathbb{E}_{s,a,s',a' \sim D}[(r(s,a) + \gamma \hat{V}_\psi(s') - Q_\phi(s,a))^2]$. The corresponding policy is extracted using advantage-weighted behavior cloning, which also avoids querying out-of-sample actions ($\pi_\theta) = \mathbb{E}_{s,a,s',a' \sim D}[(Q_\phi(s,a) - V_\psi(s)) \log \pi_\theta(a|s)]$.

We consider no prior knowledge about the task in the online RL setting. An RL agent is expected to interact with a simulated environment and leverage the collected experience to learn the task by maximizing a hand-designed reward function. Neural pathways can be integrated into any online RL algorithm and trained for divergent multitask objectives. To demonstrate the effectiveness of our method in the online setting, we use *Soft-Actor-Critic* (SAC) [29] algorithm in our experiment. SAC optimizes entropy-regularized policy objectives to drive an agent to a more exploratory behaviour while optimizing its policy. Entropy is used to encourage policy to do more exploratory behaviour and ensure that it does not collapse into repeatedly selecting a particular action. The entropy regularized objective function is as follows: $J(\theta) = \sum_{t=0}^T \mathbb{E}_{s_t \sim d_\pi, a_t \sim \pi_\theta}[r(s_t, a_t) + \alpha \mathbb{H}(\pi_\theta(.|s))]$, here $\alpha$ is *temperature parameter*, which controls the *entropy*, $\mathbb{H}$ parameter, hence the stochasticity of the

policy. It determines the relative importance of the entropy term against the reward. The conventional objective for policy gradient is recovered when $\alpha \to 0$. SAC learns a $Q$-function using an entropy-regularized objective called a soft Q-function. The soft Q-function parameters are trained to minimize the following objective:$J(\phi) = \mathbb{E}_{(s,a) \sim D}[(Q_\phi(s_t, a_t) - r(s_t, a_t) + \gamma \mathbb{E}_{s \sim \rho^\pi, a \sim \pi_\theta}[Q_{\phi'}(s_{t+1}, a_{t+1}) - \alpha \log \pi_\theta(a_{t+1}|s_{t+1})])^2]$.

## C  Additional Implementation Details

### C.1  Libraries

We run our algorithm in PyTorch-1.9.0 [58] and use following libraries: Soft-Actor-Critic (SAC) [95], Implicit Q-learning (IQL) [86], Single-shot pruning (SNIP) [1], official BCQ [22], RiGL and Rlx2 [82] implementation. In our MetaWorld experiments, we utilized the commit with the following commit-id: https://github.com/rlworkgroup/metaworld/commit/af8417bfc82a3e249b4b02156518d775f29eb289, maintaining consistency with the setup employed for benchmarking as detailed in [77].

### C.2  Offline data collection

We use the `Halfcheetah` control environments proposed in [85] and train Soft-Actor-Critic (SAC) [29] online for 1 million time steps and collect 1k expert trajectories.

For `Quadrupod` tasks we utilize the environment and trained agents from [75] and collected 1000 trajectories for each task. During data collection we find it important to use stochastic policy to add data diversity, otherwise, every trajectory follows the exact consecutive states, actions and rewards sequence due to the deterministic nature of the environment transition function.

We train SAC [29] on MetaWorld [98] environments for 3 million steps with a training batch size of 1024 samples. Similar to [77] we truncate the episode for 150 steps. We collected 1k trajectories for each environment, where we take sample action of the normal distribution rather than the mean to have diversity in the training sample. Since we are not learning a goal-conditioned algorithm, we need to keep this goal for the experiment.

### C.3  Hyper-parameter

In table 4 we present the network hyper-parameters of different algorithms that are used in this work. In *online* MetaWorld experiments, we deviate from the standard procedure and instead adopt the SAC multitask hyperparameters suggested in the benchmark [77] for fair comparison. Specifically, we employ neural networks with three layers, each containing 400 hidden units, and utilize a mini-batch size of 128.

Table 4: Hyperparameter of the network architecture used to train and evaluate offline and online RL experiments.

|  | Hyper-parameter | BCQ | IQL | SAC |
|---|---|---|---|---|
| hyper-parameter | Optimizer | Adam | Adam | Adam |
|  | Critic learning rate | 1e-3 | 1e-3 | 1e-3 |
|  | Actor learning rate | 1e-3 | 1e-3 | 1e-3 |
|  | Mini-batch size | 256 | 256 | 256 |
|  | Discount factor | 0.99 | 0.99 | 0.99 |
|  | Target update rate | 5e-3 | 5e-3 | 5e-3 |
|  | Policy update frequency | 2 | 2 | 2 |
| Architecture | Critic hidden dim | [400, 300] | [1024, 1024] | [256, 256] |
|  | Critic activation function | ReLU | ReLU | ReLU |
|  | Actor hidden dim | [400,300] | [1024, 1024] | [256, 256] |
|  | Actor activation function | ReLU | ReLU | ReLU |
|  | VAE hidden dim | [750, 750] | – | – |
|  | Value hidden dim | – | [1024, 1024] | – |

### C.4  Simulated environments

**Multitask** `HalfCheetah:`  We train `HalfCheetah` for five different kinds of task [17, 63] where it needs to (i) run forward, (ii) run backward, (iii) jump, (iv) jump while running forward and (v) jump while running backward. It is important to note that the gait movements of these tasks are very diverge.

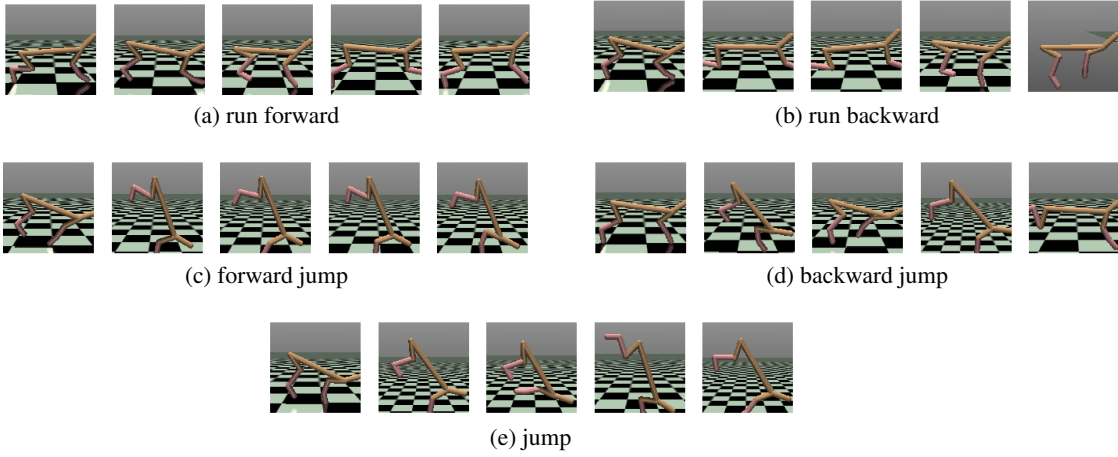

(a) run forward

(b) run backward

(c) forward jump

(d) backward jump

(e) jump

Figure 7: Snapshot of trained policy for `Halfcheetah` multitasks.

**Constrained Velocity `HalfCheetah`:** We consider another variation of the `HalfCheetah` environment where tasks are rather similar and need to go forward where we constrain the velocity of the `HalfCheetah` to six different target values from $0.5$ to max speed $3.0$ [17, 63].

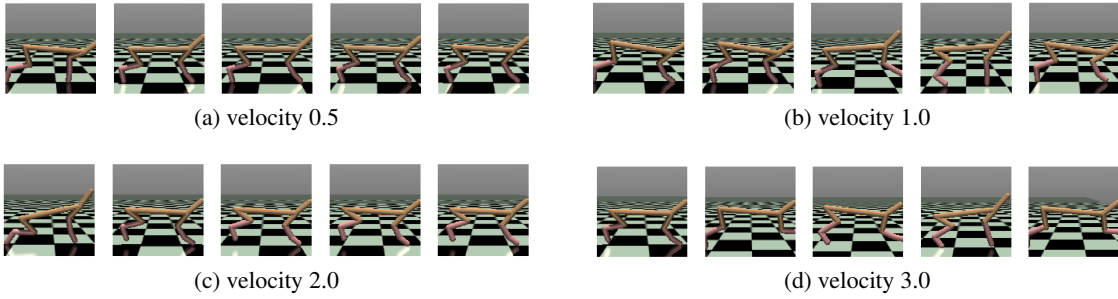

(a) velocity 0.5

(b) velocity 1.0

(c) velocity 2.0

(d) velocity 3.0

Figure 8: Snapshot of trained policy for `Halfcheetah` under four different constrained velocity.

**Multitask `Quadrupod`:** Multitask `Quadrupod` consists of (i) run forward, (ii) run backward, (iii) hopturn: hop and turn to left followed by turning back to the initial position and (iv) sidestep: take a step left and take a step to the right to the initial position. This simulated environment is commonly used in simulation to real-world transfer experiments [40, 61, 94].

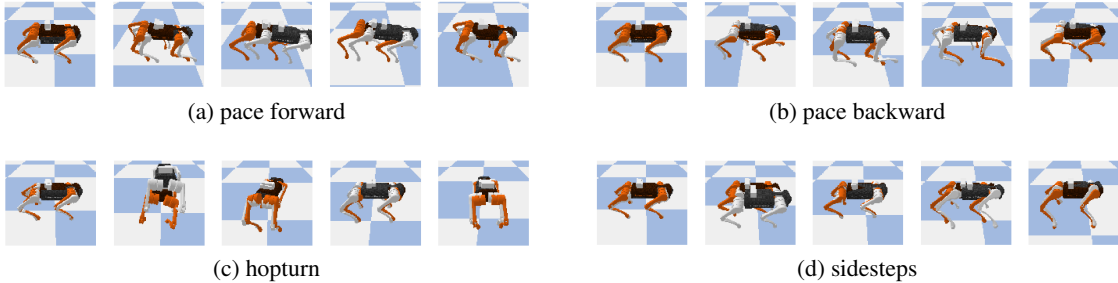

(a) pace forward

(b) pace backward

(c) hopturn

(d) sidesteps

Figure 9: Snapshot of trained policy for `Quadrupod` multitasks.

**MetaWorld:** [98] is a set of robotic manipulation tasks for benchmarking multitask-learning and meta-learning. In this paper, we consider the MT10 benchmark from MetaWorld, where we have 10 diverse tasks, evaluated with the mean success rate over 10 tasks. At each trial, the agent gets a binary score based agent's success in reaching an expected goal state.

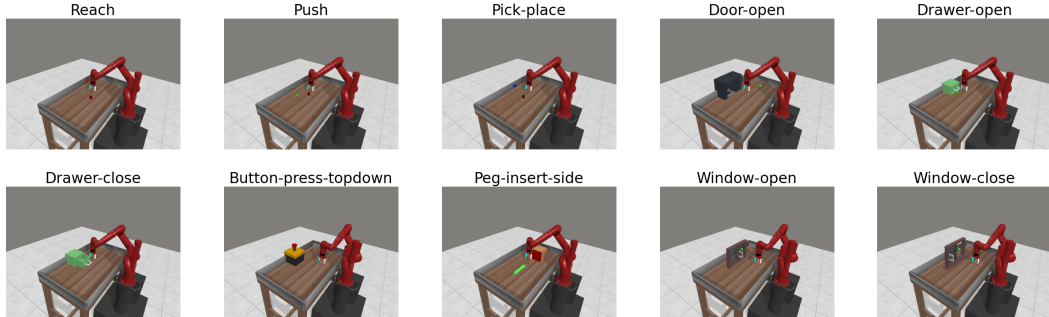

Figure 10: Snapshot of MetaWorld MT10 tasks.

## C.5 Performance evaluation

Each task in a multitask experiment is trained with an equal number of gradient updates. We evaluate performance every 5000 gradient updates, with each evaluation reporting the average episodic return and standard deviation, along with a 95% confidence interval, calculated over 10 episodes. Our results are reported over multiple seeds of the simulator and the network initialization.

For normalized performance comparison in offline RL, we normalize the episodic return using standard proposed metric from [21]: Normalized score $= \left( \frac{\text{score - random score}}{\text{expert score - random score}} * 100 \right)$. We plot the mean normalized score of multiple seeds with $100\%$ confidence interval for all our experiments.

For MetaWorld (both offline and online) we evaluate the percentage of success following the Meta-World benchmark [77].

For the Gym environment, we conduct experiments for seeds 0-4 (if not stated otherwise). We run all MetaWorld experiments for seed 10 seeds (0-9) and report the percentage of success after evaluating 100 episodes.

---

**Algorithm 2** Multitask Offline Training

---

    # Find the Task Specific Weight Masks:
    ▷ $m_1, m_2 .. m_N =$ FindPathways $\left( D_1, D_2, ... D_N \right)$; using Eq.(1).
    #Initialize a model for each task and consider asynchronous gradient update:
    ▷ Initialize local-model, global-model

    #Training loop:
    **for** Training steps ($n$ Tasks in parallel) **do**
        #sync-weight
        ▷ local-model = mask$\left($global-model$\right)$ # $L(\theta * m_n)$; mask the $n^{th}$ task-specific weights
        ▷ Sample $(s, a, s') \sim D_n$
        ▷ local-model.loss()
        ▷ local-model.backward() # $\nabla_{\theta \odot m_n} J(D_n)$ ; compute gradient of masked weights
        ▷ sync-gradient$\left($local-model, global-model$\right)$
        ▷ global-model.optimizer.step()
    **end for**

    #Evaluation loop:
    ▷ local-model = mask$\left($global-model$\right)$ # sync-weight
    ▷ Evaluate( local-model)

---

# D  Additional Results

## D.1  Online RL Expeiments

**Empirical Validation of Many Lottery Ticket Sub-networks:** Empirically, we have found that, if we keep configuring the pathway throughout the training, it never optimizes to a fixed set of parameters. A certain % of the pathway is always changing. Pruning literature hypothesizes [18] existence of a *set of lottery sub-networks* (where each subnetwork is expected to perform the same as the dense network). Therefore, we conjecture that in online RL, without the stopping criteria, it ends up switching among these sub-networks and that leads to high variance in performance

Here, we empirically want to show that after the warm-up phase, we can find many *lottery sub-networks*, when optimized separately can converge to similar performance. In the online setting, after the *warm-up* phase we stop re-configuring pathways any further. However, there is no theoretical justification as to whether or not we have converged to an *optimal pathway*. Pruning literature hypothesizes that many sub-networks can lead to equivalent per-

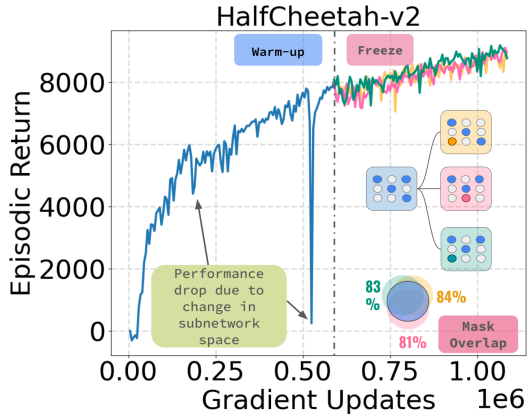

Figure 11: After the warm-up phase, we sample three sub-networks (highlighted in yellow, pink, and green). These sub-networks exhibit a certain percentage of change in the parameter space from the prior sub-network (blue). We illustrate the overall percentage change using the Venn diagram. The similarity in the learning curves supports the existence of multiple equally effective lottery sub-networks.

formance [18]. We back this up empirically in Figure 11. After the warm-up phase, we sample three different trajectories and, using Eq. 4 obtain three subnetworks. By using different seed values and sampling stochastic actions, we ensure the subnetworks are distinct. These sub-networks exhibit a certain percentage of change but *highly overlapping* in the parameter space from the prior sub- network. We highlight the overlap using the Venn diagram in the Figure 11. Training these subnetworks separately yields similar performance. The similarity in the learning curves supports the existence of multiple equally effective lottery sub-networks. Therefore, after the warm-up phase, it is justified to use any of these sub-networks and use it for the remainder of the training. Otherwise, as observed in the warm-up phase, we will see an *occasional* drop in performance due to a sudden change in the sub-network parameter space.

**Ablation on the baselines:** As reported in [82], the gradient-based topology evolution methods (RiGL [27] and Rlx2 [82]) depend on finding unique actor-critic sparsity ratio for different tasks to reach the performance of the dense network. In this work, we evaluate all the methods at the sparsity setting of 95%. We use the RiGL and Rlx2 implementations as reported in [82]. We rerun the experiments on Ant-v2 control task to further assess the degree of dependency. In Fig 12, we observe the performance for both methods drop drastically (orange curve) when we set the sparsity of actor and critic network to 95%.

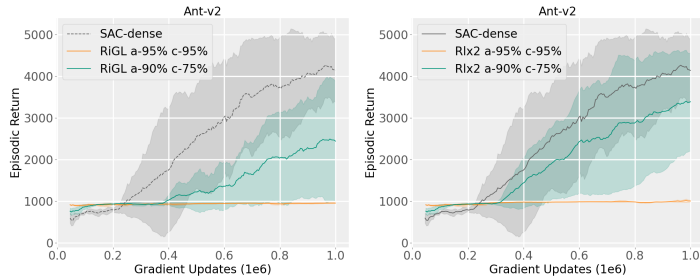

Figure 12: Learning curve of RiGL (left) and Rlx2 (right) under different sparsity levels. We compare the performance of optimal sparsity level (actor 90% and critic 75% in green) presented in [82] with the performance at 95% sparse network ((for both actor and critic in orange).

**Compare baseline performance:** The comparative analysis, depicted in Figure 13, further solidifies our assertion. This analysis is conducted at a sparsity level of 95% on the learning curves, comparing SAC-dense performance with SAC-sparse (95%) trained using DAPD, RiGL, and Rlx2 across continuous control tasks: `HalfCheetah-v2`, `Walker2d-v2`, `Hopper-v2`, and `Ant-v2`. The models are trained for 1 million steps and evaluated every 5000 steps. Episodic return is averaged over 10 evaluations across seeds 0-4.

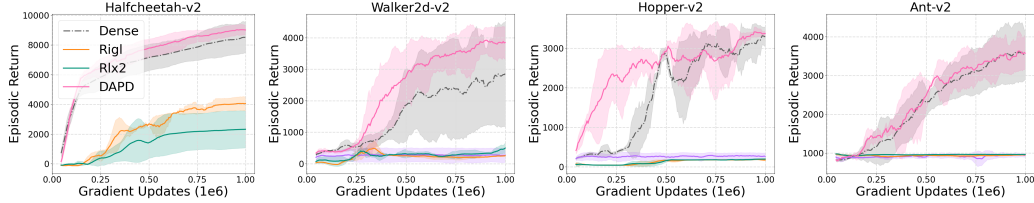

Figure 13: At **95% sparsity**, we compare the performance of DAPD with baselines on MuJoCo control tasks.

**Performance comparison across different sparsity levels:** For further validation, we present the final performance across various sparsity levels in Fig 14, where DAPA consistently prior superior performance by a large margin and even exceeds the dense network in most of the experiments.

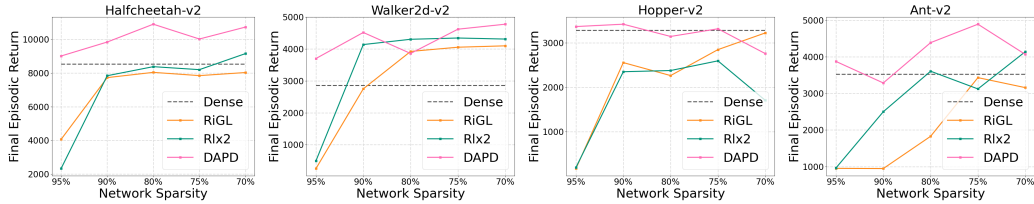

Figure 14: Performance comparison of DAPD with baselines across various network sparsity levels on MuJoCo control tasks.

**Sample Efficiency** : Here we provide the complete results on the sample efficiency of DAPD compared to the SAC-dense model in MuJoCo continuous control tasks.

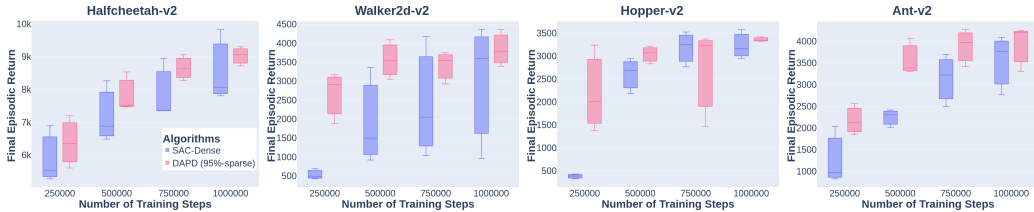

Figure 15: Sample efficiency comparison of SAC-sparse (95%) network using DAPD with the SAC-dense counterpart. We average episodic return over the 10 evaluations over 5 seeds at different training steps.

**DAPD hyper-parameter tuning:** We compare the performance of DAPD varying the warm-up phase threshold ($TH$) and length of the moving average over the scoring function $K$ in Table 5. In the majority of cases, we observe performance surpassing that of the SAC-dense network. Moreover, with the appropriate configuration of $TH$ and $K$ values, we achieve a marginal improvement, exceeding the SAC-dense performance. However, we find in certain experiments, that the length of the moving average over the scores can stabilize the performance and can be very effective. We further validate a set of experiments on `HalfCheetah`-tasks [17, 63]. We showcase the importance of tuning the parameter $K$ in Table 6. Subsequently, we compare the performance with RiGL, Rlx2, as well as with Tiny-SAC (a dense network with 32 hidden units). We normalize (on a scale of 100) the episodic return with the single-task SAC-dense performance.

For MetaWorld, we utilize the success of reaching to goal state as the warm-up threshold $TH$ and use moving average K=1 for our experiments.

Table 5: Performance comparison of DAPD conditioning the warmup phase, $TH$ and the $K$-length moving average over scores in Eq. (4). The evaluation is performed after 1 million training steps, and the mean and standard deviation are computed based on 10 evaluations across seeds 0-4. We **bold** the performance where we exceed the SAC-dense and <span style="background-color:#add8e6">highlight</span> the best performance.

| Environment | Warmup Episodic Return | Moving Average K | | | SAC-dense |
| --- | --- | --- | --- | --- | --- |
| | Threshold (TH) | K=1 | K=5 | K=10 | |
| HalfCheetah-v2 | TH=6k | **8581.65 ± 1239.29** | 7762.61 ± 1409.99 | 8139.22 ± 373.86 | 8568.1 ± 1043.56 |
| | TH=7k | **8767.34 ± 970.99** | 8528.14 ± 596.87 | 7695.5 ± 297.06 | |
| | TH=8k | **9028.02 ± 276.31** | **8661.59 ± 141.25** | 8365.33 ± 217.94 | |
| Walker2d-v2 | TH=2k | **3760.02 ± 79.99** | **3492.1 ± 450.99** | 2623.14 ± 1359.74 | 2972.49 ± 1691.47 |
| | TH=3k | **3246.02 ± 158.14** | **3660.02 ± 406.87** | 1719.17 ± 1233.02 | |
| | TH=3.5k | **3810.9 ± 529.2** | **3846.3 ± 459.82** | **3564.79 ± 354.65** | |
| Hopper-v2 | TH=2k | **3359.88 ± 46.57** | 2844.01 ± 518.6 | 2269.41 ± 1293.42 | 3228.5 ± 301.88 |
| | TH=3k | **3289.11 ± 315.19** | 2865.94 ± 874.77 | 3005.91 ± 488.89 | |
| | TH=3.5k | 3202.11 ± 330.27 | 2567.26 ± 818.61 | 2723.71 ± 619.53 | |
| Ant-v2 | TH=2k | 2698.21 ± 941.56 | 2845.63 ± 517.76 | 2442.02 ± 244.73 | 3538.22 ± 654.76 |
| | TH=3k | **3576.48 ± 416.99** | **3916.65 ± 502.82** | 3297.23 ± 984.44 | |
| | TH=3.5k | 2232.0 ± 1054.63 | **3868.91 ± 786.66** | 3032.03 ± 749.78 | |

Table 6: Performance ablation of DAPD on various `HalfCheetah`-tasks. We present the normalized performance compared to SAC-dense network.

| Environments | DAPD | | | | |
| --- | --- | --- | --- | --- | --- |
| | K=10 | K=5 | K=1 | DAPD w/o freeze | DAPD w/o warm-up |
| Halfcheetah Forward | **98.57 ± 0.81** | **98.88 ± 5.90** | 97.97 ± 0.6 | 71.33 ± 26.14 | 26.89 ± 10.34 |
| Halfcheetah Backward | 82.71 ± 6.86 | 87.53 ± 9.4 | 76.64 ± 12.60 | **89.91 ± 10.36** | 12.13 ± 3.8 |
| Halfcheetah Jump | **93.63 ± 6.87** | 66.60 ± 32.25 | 84.44 ± 12.60 | 14.10 ± 39.48 | 0.98 ± 3.2 |
| Halfcheetah Foward Jump | **98.54 ± 0.80** | 90.53 ± 12.9 | 89.62 ± 13.63 | 66.00 ± 46.80 | 25.54 ±8.5 |
| Halfcheetah Backward Jump | **88.85 ± 0.12** | 85.67 ± 3.9 | 65.55 ± 3.26 | 80.24 ± 9.5 | 11.89 ± 3.7 |
| Overall | **462.33 ± 15.49** | 429.16 ± 59.16 | 414.17 ± 42.75 | 293.38 ± 13.23 | 77.45 ± 29.70 |
| Improvement | **6x** | **5.5x** | **5.3x** | **3.8x** | - |

**Algorithmic Generalization:** To assess the general applicability of our method across various RL policies, we tested it on Proximal Policy Optimization (PPO) [70] for continuous control comparing our method's performance against a dense network in single-task settings. Our results in Fig 16 demonstrate that our approach enhances performance for on-policy actor-critic methods like PPO. We provide the learning curve over 3 seeds below:

**Domain Generalization:** To demonstrate the generality of the approach and to check performance in other domains, we provide the performance of DAPD in three pixel-based Atari environments. We explored scenarios without any assumption about the expected return and explored the possibility of updating the mask periodically. We conducted experiments using the DQN [47], updating the mask every $L$ gradient steps. We report the final performance in Table 7 after 10 million gradient steps, averaging over 3 seeds, with the mask being updated every L=1 million steps.

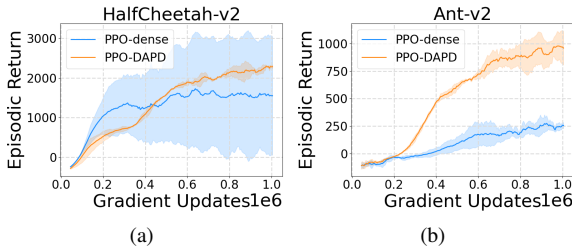

(a)    (b)

Figure 16: To prove algorithmic generality, we compare the PPO baseline performance with applying DAPD on (a) HalfCheetah-v2 and (b) Ant-v2 tasks

## D.2 Online Multitask Training and Addressing Gradient Interference

Even though training pathways (each constituting 5% of the neural network) compartmentalize the neural network parameter space, due to gradient interference, optimizing pathways while learning

Table 7: Performance of DQN Algorithms on Various Environments

| Environment | DQN-dense (mean ± std) | DQN DAPD (mean ± std) |
|---|---|---|
| DemonAttack-v4 | 17670.33 ± 2829.91 | **20803.33 ± 3273.07** |
| BreakoutNoFrameskip-v4 | 346.66 ± 12.21 | **384.0 ± 15.80** |
| PongNoFrameskip-v4 | **20.36 ± 0.58** | 19.09 ± 0.77 |

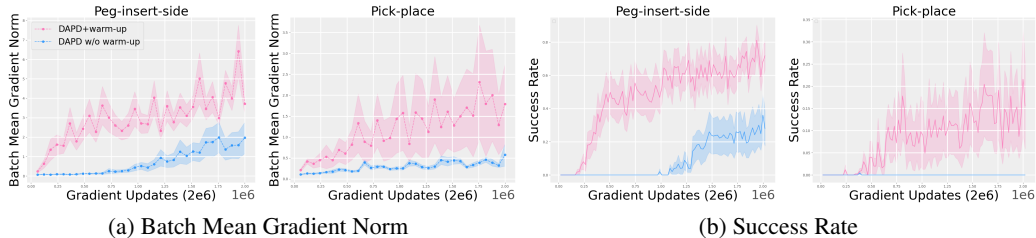

     (a) Batch Mean Gradient Norm            (b) Success Rate

Figure 17: When we try to *configure pathways without warm-up phase* (blue), (a) parameters of the harder tasks do not change from the values at initialization due to small changes in gradients. This results in unsuccessful task learning and can be seen in (b). We find an improvement in performance including *warm-up phase* (pink).

good policy for each task becomes tricky. Notably, significant parameter sharing among tasks makes it difficult to entirely avoid such interference.

Our experiments with MetaWorld [98] revealed that DAPD rapidly optimizes pathways for easier tasks, but for more challenging tasks, we observed only minimal changes in gradient updates. To monitor parameter changes, we computed the gradient norm $||\frac{dw}{dL}||$ of the pathway parameters of the SAC-actor network for a batch of samples.

We observe an interesting correlation between the change in gradient norm in Fig 17(a) and the performance in Fig 17(b). In Fig 17(a), initial experiments *without warm-up* phase (blue) showed little to no change in gradient norm for task-specific parameters. Whereas deploying *DAPD with warm-up* phase (pink) escapes the stagnation and contributes to an improvement in the success rate, as illustrated in Fig 17(b). **It is important to note that while a change in the gradient norm is indicative of some updates occurring during training, it does not necessarily guarantee improved performance**. Comparing the blue and pink curves in 17(a), we can conclude that having a warm-up phase, to configure pathway, is helping to diverge from random weight initialization, which can be useful for overcoming local minima or exploring the solution space. A reflection of improvement we see in the success rate in Fig 17(b).

For the MetaWorld experiment, we employ the warm-up phase for 10,000 gradient steps (constituting 0.05% of the training) to learn independent task-specific pathways. For each task network parameter initialization is identical. At the end of this warm-up stage, we average the weights of overlapping parameters and keep the pathways fixed for the remainder of the training. We conduct parallel online data collection, and during training, we accumulate gradients of shared parameters.

## D.3 Offline RL Experiments:

For the offline RL setup, when designing experiments, we focus on two criteria. *First*, how does our method perform compared to the performance of *single task experts* and *second*, how does our method perform when compared to a common baseline. When comparing to an expert, we carefully select a range of multitask benchmarks ensuring that our method (1) is tested with a diverse range tasks (Multitask `HalfCheetah`) (2) can yet be performed in a controlled tasks that are similar (`HalfCheetah` Constrained Velocity) in nature and (3) can master skills that can easily be transferred in real world applications (Multitask `Quadrupod`). To compare to common baseline algorithms, we evaluate on the (4) MetaWorld Multi-Task benchmark.

**Evaluation method:** For evaluation, we compare with expert performance and normalize the episodic return using standard offline evaluation metric [21], *normalized score* $= \left(\frac{\text{score - random score}}{\text{expert score - random score}} * 100\right)$, where *random score* is generated by unrolling a randomly initialized policy and averaged over 100 episodes. A score of 100 represents the average returns of a domain-specific expert.

We conduct offline MetaWorld experiments with 10 different seeds, employing 100,000 gradient updates for each task. The evaluation of our trained agent is performed 100 times for each task, providing a more accurate assessment of its performance across all tasks.

### D.3.1 Performance Compared to Single-task Expert:

Each task in a multitask experiment is trained with an equal number of gradient updates. We compute the average episodic return over 10 episodes of the trained agent. Our results in Table 8 are reported over seeds 0-4 of the Gym simulator and the network initialization. We run the `Halfcheetah` experiments for 1M and `Quadrupod` for 500k gradient updates. Our proposed method achieves expert-like performance for most of the tasks in 3 sets of experiments.

Table 8: Individual task normalized score (performance compared to *single task expert*) `HalfCheetah` and `Quadrupod` tasks.

| Experiment | Tasks | PD | |
|---|---|---|---|
| | | BCQ | IQL |
| `HalfCheetah` multitask | Forward | $99.68 \pm 0.04$ | $99.98 \pm 0.04$ |
| | Backward | $96.51 \pm 0.19$ | $98.68 \pm 0.83$ |
| | Jump | $71.38 \pm 9.46$ | $74.99 \pm 20.1$ |
| | Forward-Jump | $85.25 \pm 0.54$ | $87.35 \pm 27.0$ |
| | Backward-Jump | $83.3 \pm 1.46$ | $87.34 \pm 14.23$ |
| | **Overall** | $87.20 \pm 2.34$ | $89.67 \pm 12.44$ |
| `HalfCheetah` Constrained speed | Velocity 0.5 | $100.04 \pm 0.03$ | $99.1 \pm 2.18$ |
| | Velocity 1.0 | $100.07 \pm 0.03$ | $100.05 \pm 0.04$ |
| | Velocity 1.5 | $100.03 \pm 0.05$ | $99.99 \pm 0.06$ |
| | Velocity 2.0 | $100.07 \pm 0.01$ | $100.07 \pm 0.04$ |
| | Velocity 2.5 | $100.02 \pm 0.04$ | $100.04 \pm 0.03$ |
| | Velocity 3.0 | $99.96 \pm .07$ | $100.01 \pm 0.03$ |
| | **Overall** | $100 \pm 0.038$ | $99.88 \pm 0.40$ |
| `Quadrupod` multitask | Forward | $116.19 \pm 0.14$ | $116.92 \pm 0.51$ |
| | Backward | $110.98 \pm 1.21$ | $112.93 \pm 1.05$ |
| | Hopturn | $122.43 \pm 0.47$ | $123.33 \pm 0.29$ |
| | Sidestep | $111.8 \pm 0.96$ | $113.69 \pm 0.43$ |
| | **Overall** | $115.35 \pm 0.70$ | $116.71 \pm 0.57$ |

**Performance curve of Halfcheetah Multitask**  Fig. 18 presents the performance of five different `Halfcheetah` tasks: (1) run forward, (2) jump while run forward, (3) run backward, (4) jump while run backward and just(5) jump. Snapshots of the different tasks are demonstrated in Fig. 7.

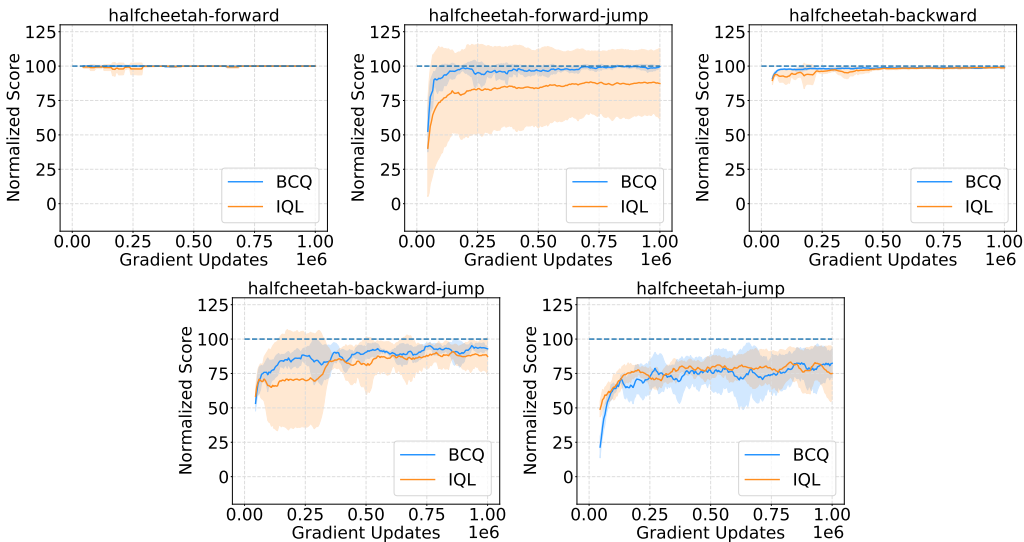

Figure 18: Performance (Normalized Score) plot of BCQ and IQL on `HalfCheetah` multitask - (Forward, Forward-Jump, Backward, Backward-Jump, Jump) trained with PD.

**Performance curve of Halfcheetah constraint goal velocity**   In Fig. 19 we present the perfor-
mance of six (6) constrained goal velocities for `Halfcheetah` running forward, where we vary the
constrained from 0.5 to max speed 3.0 with a constant increase of the speed. The tasks are trained for
500k gradient updates and evaluated every 5000 gradient updates. Snapshots of the different tasks
are demonstrated in Fig. 8.

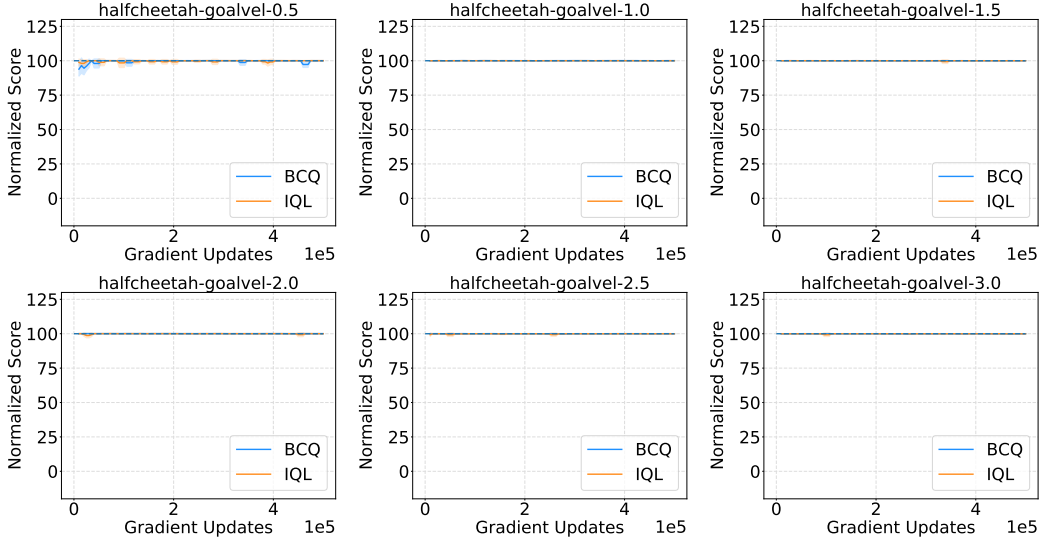

Figure 19: Performance (Normalized Score) plot of BCQ and IQL on `HalfCheetah` with constrained velocity
trained with PD.

**Performance curve Multitask Quadruped Robot**   We use the `Quadruped` robot to perform four
(4) tasks: hopturn, pace forward, pace backward, sidestep. We use [75] as our expert to collect 500k
(1000 trajectory) expert data for each task. The environment has a deterministic transition function
and thus we take sample action from normal distribution instead of taking distribution mean to get a
diverse set of training datasets. The performance curve with PD for individual tasks in shown in Fig.
20. Snapshots of the different tasks are demonstrated in Fig. 9.

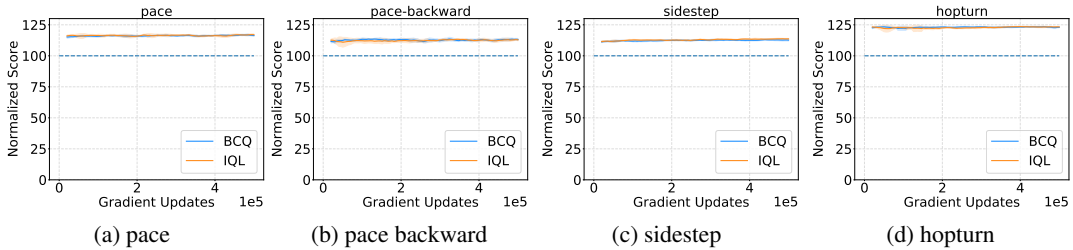

(a) pace        (b) pace backward        (c) sidestep        (d) hopturn

Figure 20: Performance (Normalized Score) plot of BCQ and IQL on `Quadrupod` multitasks with PD.

### D.3.2   Performance Compared to Multitask Baseline:

**Performance on MetaWorld**   In Table 9 we report mean performance of individual MT10 envi-
ronments over 10 seeds each evaluated for 100 episodes. We also compare individual performance
with *Conservative data sharing* (CDS) [96], an offline-multitask learning algorithm that focuses on
training based on task similarity, also reports its MetaWorld experiments on 4 environments and
reported performance over 6 seeds with 95% confidence interval.

**Offline Multitask Under Mixed Data Distribution**   Data distributional shift is a common problem
in offline RL since the data set is collected by some unknown policy and most of the offline methods
try to guarantee the best performance under such circumstances. *But data distribution shift from a*

Table 9: Individual task performance (success-rate) comparison of MetaWorld.

| MT10 | PD | | CDS [96] |
|------|------|------|------|
| | BCQ | IQL | |
| reach | $100.0 \pm 0.0$ | $93.0 \pm 22.14$ | - |
| push | $100.0 \pm 0.0$ | $100.0 \pm 0.0$ | - |
| pick-place | $100.0 \pm 0.0$ | $95.0 \pm 15.81$ | - |
| door-open | $100.0 \pm 0.0$ | $100.0 \pm 0.0$ | $58.4 \pm 9.3$ |
| drawer-open | $100.0 \pm 0.0$ | $85.0 \pm 33.75$ | $57.9 \pm 16.2$ |
| drawer-close | $100.0 \pm 0.0$ | $100.0 \pm 0.0$ | $98.8 \pm 0.7$ |
| button-press-topdown | $100.0 \pm 0.0$ | $100.0 \pm 0.0$ | - |
| peg-insert-side | $100.0 \pm 0.0$ | $100.0 \pm 0.0$ | - |
| window-open | $100.0 \pm 0.0$ | $100.0 \pm 0.0$ | - |
| window-close | $100.0 \pm 0.0$ | $100.0 \pm 0.0$ | - |
| door-close | - | - | $57.2 \pm 16.2$ |
| Overall | $\mathbf{100 \pm 0.0}$ | $\mathbf{97.3 \pm 7.17}$ | $72.0 \pm 26.2$ |

*mixture of policies does not affect discovering neural pathways in the offline setting as it does in online RL.*

In an offline setting, the *data distribution is fixed* and known prior (no further changes in data distribution during the training) and thus we can discover the neural pathway beforehand at a single shot using Eq (1). That is, in the Offline RL setting we already know the dataset we want to learn the behaviour from. Hence, the saliency criterion we use can generate the appropriate pathway.

To support our claim, we run further experiments in offline RL multitask under different data distribution shifts and compare the performance with the baseline as shown in Fig. 6. We use mixed and imperfect datasets where (i) **medium**: dataset collected from suboptimal agent trained for 300k gradient steps, (ii) **medium-expert**: mixing equal amounts of expert demonstrations and suboptimal data and (iii) **expert-replay**: recording all the sample observed by the agent during training and represents a dataset generating from a mixture of many distributions. To handle this mixture of distribution we take a larger batch of sample (x10) to evaluate important parameters.

### D.4 Overlapping neural pathways

In this work, we consider choosing the neural pathways for each task independently from one another. We find a large % of the pathways overlap, reducing the number of trainable parameters. The pathway configuration is dictated by three factors: (1) different learning objectives (i.e. offline, online), (2) scoring function (i.e. equation 1, 4), and (3) training samples used to optimize the scoring function. In Fig. 21(h) we see the % of active weights (5% of the actual network) in the policy that is shared with other tasks for IQL multitask-offline training. The diagonal of these symmetric matrices represents the number of unique weights that are optimized only for one task, and the columns represent the % of weights each task shares with others. As we see in Fig. 21(h), for all tasks, the % of weights that are optimized for just one task is very low. For *push*, only 5.5% of the *active weights* are uniquely trained for the task. We also show similar matrices for other experiments in the Appendix (Fig. 21), where we found the percentage of overlapping weights does not correlate with the task similarity. This disproves a conventional class of thinking that the success of multitask in neural networks depends on relevant data sharing [96, 16] or similar task training [77, 93]. Rather, neural networks are capable of learning multiple tasks simultaneously as long as the neurons are wired properly.

In Figure 21 we demonstrate the percentage of active weights (5% of actual network) in policy network that are shared with other tasks for different experiments and for different offline RL algorithms. The diagonal of these symmetric matrices represents the number of unique weights that are optimized only for one task. The columns represent the percentage of weights the task shares with others.

It is important to note, in Figure 21, for all our experiments the percentage of weights that are optimized for just one task is very low. For example, in Figure 21-*a* for the forward task, only 26% of the active weights are uniquely trained for the task. We only activate/allow 5% of the original network weights for one task after pruning. This means only 0.13% of the original network is trained uniquely to make `Halfcheetah` run forward like an expert. Also, we do not find the percentage of overlapping to correlate with the similarity of the tasks.

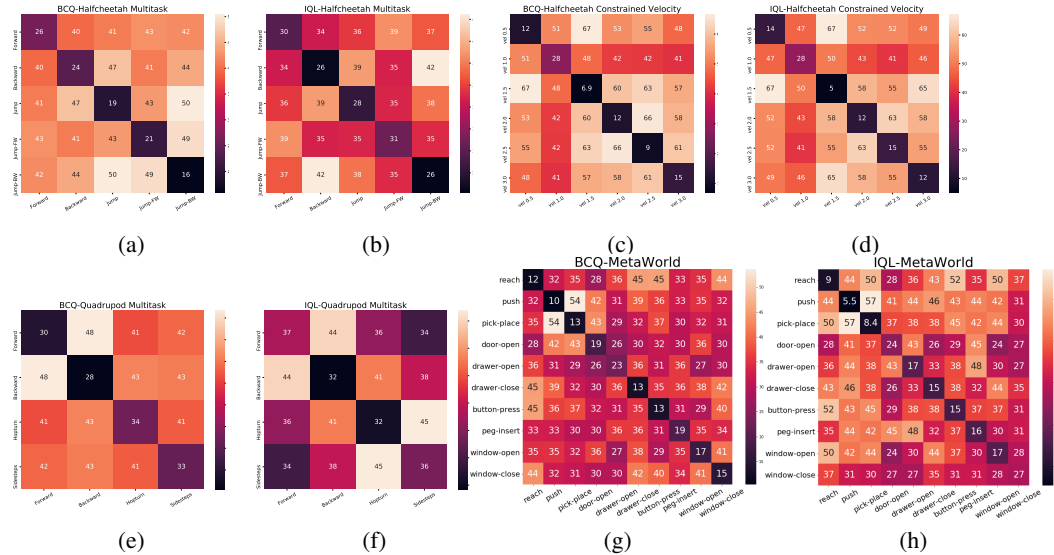

Figure 21: Shows the percentage of trainable weights are being shared among different tasks. In $a$ and $b$, we compare shared weights among five `Halfcheetah` tasks: (1) forward (F), (2) backward(B), (3) jump(B), (4) Forward while Jumping (FJ) (5) Backward Jumping (BJ). In $c$ and $d$, we compare shared weights among six `Halfcheetah` tasks where we increase forward velocity 0.5-3.0 . In $e$ and $f$, we compare four `Quadrupod` tasks. In $g$ and $h$ we show the shared weights among MetaWorld tasks.

**Overlapping Pathways Lead to Fewer Parameters:** For each task, we only use $5\%$ of the total weights of the actual network while maintaining expert-like performance for all tasks. For $N$ tasks it has an upper bound of $\frac{N}{20}x$ parameters when all the pathways are unique, but we find the pathways to overlap significantly (in Figure 21), further reducing the number of total weights. To understand how PD compacts multiple experts into a single network, in Table 10, we look at the number of policy parameters IQL requires compared to SAC with and without pruning on the individual MT10 tasks. In our experiments, the total number of network parameters for SAC and IQL are identical with $1,092,608$ parameters in the policy network. With MT10 the number increases $10\times$. Due to the inherent shared structure of neural pathways, our method requires only $1.82\%$ (averaged over 10 seeds) of the original network. To get equivalent utility we have to train ten SAC agents with $98.18\%$ pruned networks, which is not possible using any existing pruning techniques without losing performance [18, 38, 90, 84]. As seen in [90] at $98\%$ reduction of the network weights pruning techniques suffers from significant performance drop due to layer collapse. Even when we compare to $95\%$ pruned SAC trained on 10 different individual tasks ((B) in table 10), we have $36.4\%$ reduction in actor parameters.

Table 10: Comparison of the number of parameters required in MetaWorld for different methods.

| | MT-10 trained as separate tasks | | MT-10 multitasks trained using **PD** |
|---|---|---|---|
| | (A) w/o pruning | (B) 95% pruned | (C) 95% pruned |
| Parameter counts (Policy/Actor Network) | 11 Million (10,926,080) | 546,304 | $198,956 \pm 7,636$ ( $1.82 \pm 0.07\%$ of A) |

# E    Limitations and Future Work

**Significance of sparsity in performance:** We chose a 95% sparsity level for the parameters because previous works in supervised learning [38] and offline RL [3] have demonstrated that using the saliency criterion in Equation (1) [38] achieves reliable performance, comparable to that of dense baseline models. We ran an experiment for 1 million steps with SAC on the `HalfCheetah-Forward` [17] at three different sparsity levels to justify our choice. More sparsity leads to performance loss (lower episodic return). Increasing the sparsity further can provide a more energy-efficient solution.

Table 11: Performance comparison of SAC-DAPD under different sparsity levels.

| Sparsity level used | 99% | 97.5% | 95% |
|---|---|---|---|
| HalfCheetah-Forward | -56.67 ± 44.91 | 1301.59 ± 424.91 | **8528.71 ± 664.85** |

**Empirical Pathway Convergence:**   In the online setting, we use the *warm-up* phase to stop re-configuring pathways any further. However, there is no theoretical justification as to whether or not we have converged to an *optimal pathway*. Pruning methods hypothesize that there are many sub-networks that can lead to equivalent performance [18]. The algorithms developed within pruning methods mostly provide empirical guarantees through performance but none of them provide a theoretical guarantee of superior performance. However, we would like to point out that having heuristic scheduling as a stopping criterion is commonly used in the iterative pruning literature [65, 19]. We empirically found this simple stopping criterion to be effective throughout the experiments discussed in the paper, which suggests the found pathway leads to stable performance.

**Pathway Overlap and Task Relevancy:**   The pathway overlapping depends on various factors, such as weight initialization, data collection, and policy improvement. As we see in Figure 21 there is no clear relationship between the tasks and pathways overlap. Further research is required to interpret the task relation.

**FLOP and Energy Consumption**   Computing energy consumption for ML research has been debatable. There has been ongoing research on how to properly estimate the carbon footprint [31, 59]. In this paper, we present the *theoretical gain* in energy efficiency. Using SpMM-aware hardware [42, 99, 25, 6, 49, 55, 52, 54] and software [53, 39, 62], the reduction in FLOP counts decreases the number of computations in SpMM. This reduction in computations consequently lowers energy costs, resulting in a proportional relationship between FLOP count and energy cost in *Joules* i.e. FLOPs $\propto$ Joules.

**Real World Application:**   Learning multitask using multiple pathways is more favourable to "real-world applications". We support our claim by showing that PD requires the lowest FLOP counts by $20x$ fold compared to offline (Table 3) and online (Table 2) baselines. A lower FLOP count is proportional to faster inference. However, to completely exploit the benefits of sparse models, adaptations are required in lower-level libraries that enable modern deep-learning frameworks. The latest Nvidia GPUs [54] are focusing on taking advantage of sparsity in parameters. Further hardware and software optimization in sparse training will allow RL agents to train faster and be used in low-resource, large-data-driven real-time applications. We argue that we were inspired in our explanation by the standards in the pruning literature. Hardware [30, 11] that leverages sparse weights will be able to exploit pathways and thus be favourable to resource-constrained environments such as edge devices like embedded systems, cellular phones or robotics, where deploying large models is a challenge.

